# Sketching for Distributed Deep Learning: A Sharper Analysis

**Mayank Shrivastava**
University of Illinois Urbana-Champaign
mayanks4@illinois.edu

**Berivan Isik**[*]
Google
berivan@google.com

**Qiaobo Li**
University of Illinois Urbana-Champaign
qiaobol2@illinois.edu

**Sanmi Koyejo**
Stanford University
sanmi@cs.stanford.edu

**Arindam Banerjee**
University of Illinois Urbana-Champaign
arindamb@illinois.edu

## Abstract

The high communication cost between the server and the clients is a significant bottleneck in scaling distributed learning for overparametrized deep models. One popular approach for reducing this communication overhead is randomized sketching. However, existing theoretical analyses for sketching-based distributed learning (sketch-DL) either incur a prohibitive dependence on the ambient dimension [62] or need additional restrictive assumptions such as heavy-hitters [59]. Nevertheless, despite existing pessimistic analyses, empirical evidence suggests that sketch-DL is competitive with its uncompressed counterpart, thus motivating a sharper analysis. In this work, we introduce a sharper ambient dimension-independent convergence analysis for sketch-DL using the second-order geometry specified by the loss Hessian. Our results imply ambient dimension-independent communication complexity for sketch-DL. We present empirical results both on the loss Hessian and overall accuracy of sketch-DL supporting our theoretical results. Taken together, our results provide theoretical justification for the observed empirical success of sketch-DL.

## 1 Introduction

Distributed learning is a popular framework for training machine learning models, often deployed to support large-scale deployments and to support privacy, among other systems goals [20, 58]. To this end, distributed learning generally employs both server and client devices. Most standard implementations are set up in rounds: in each round, clients participate by performing (multiple) local model updates using their local data, then share the model updates with a server. The server aggregates the collected local updates and broadcasts either the aggregate or the updated global model to the clients for the next round. As a result of the system setup, distributed learning generally requires a high communication overhead due to the frequent communication between the server and the clients [42]. Federated learning, a popular distributed learning setup, further extends the generic setting to where clients' data distribution may be non-i.i.d. and the clients may change over time, thus suffering from the same costly communication. This communication overhead has motivated research in compressing

---

[*]Part of the work done while BI was at Stanford University.

the model updates [35, 43] through sparsification [44, 73, 7, 30, 31], quantization [4, 74, 10], low-rank projection [71, 67, 68, 8], and sketching [59, 62, 50, 33, 27]. Among these, linear sketching mechanisms have attracted significant attention due to their efficient and simple integration with existing distributed and federated learning frameworks. For instance, unlike many sparsification approaches [44, 30], sketching is an unbiased operation that does not require bias correction with error feedback mechanisms [63] especially as error feedback typically increases the memory cost and the complexity of integrating differential privacy [23]. Moreover, the linearity of the sketching operation makes it compatible with secure aggregation [12, 9] as opposed to the popular `Top-r`[2] sparsification [44] or quantization methods [4] that break the linearity in aggregation.

Despite the rising interest in sketching for distributed and federated learning, the existing bounds on the convergence error of sketching scale with the ambient dimension of the model [62] (i.e., in contrast to the lower sketching dimension) which may be extremely large for modern overparameterized deep models [39, 20, 11]. This dimension dependence has been a limitation of the sketching mechanisms at scale, making them less attractive for modern deep models. We identify and demonstrate the root cause of the dimension dependence of the standard optimization convergence analysis as associated with the assumption of isotropic smoothness of the loss, commonly made in such analyses [62, 59]. Several modifications have been proposed to get rid of the dimension dependence. For instance, Rothchild et al. [59] assume that the model updates at the clients have heavy hitters and, further, apply a `Top-r` sparsification to break the dimension dependence. However, their heavy hitter assumption may not hold in general, and the `Top-r` operation introduces a bias that is, in turn, eliminated via error feedback. While their analysis requires these restrictive assumptions and modifications to eliminate the dimension dependence, they also note that they actually do not observe an adverse effectt of dimension dependence empirically even without these restrictions, indicating that the theoretical analysis of their sketching algorithm is not quite explaining the empirical success. Quoting from Appendix B.3 of Rothchild et al. [59], "However, this dimensionality dependence does not reflect our observation that the algorithm performs competitively with uncompressed SGD in practice, motivating our assumptions and analysis." Motivated by this mismatch between the existing analyses of sketching for distributed learning (which impose dimension dependence) [59, 62] and its empirical success in practice, we provide a substantially tighter analysis than the prior work and eliminate the dimension dependence in sketching without imposing any unrealistic restrictive assumptions in the setup.

Our sharper analysis avoids the global isotropic smoothness assumption and utilizes the anisotropic restricted strong smoothness (RSS) property of overparameterized deep models [6, 72, 22, 53, 52, 15], and our results are in terms of the second order geometry of the loss. We present optimization results for single-local step as well as $K$-local step distributed learning, and present bounds on communication complexity based on the optimization results. We refer the reader to Table 1 for an overview of our results. We do not restrict our analysis to any specific sketching matrix, and our results hold for any symmetric sub-Gaussian sketching matrix. For instance, for computational benefits, one can use the popular `Count-Sketch` [14] or Hadamard sketch [2] approaches, which are both examples of sub-Gaussian sketching. Our contributions can be summarized as follows:

1. We identify and demonstrate the widely used global smoothness assumption of loss functions as the root cause of the dimension dependence of sketching methods.

2. We provide a novel analysis for sketching in distributed and federated learning that eliminates the ambient dimension dependence in the convergence error.

3. We are the first to break this dimension dependence without making restrictive assumptions such as the heavy hitter assumption by Rothchild et al. [59].

4. We are again the first to do this without a `Top-r` sparsification step in the framework which would have required additional measures to eliminate the bias.

5. We are the first to use more precise second-order properties of the loss of deep models, e.g., approximate restricted strong smoothness, eigenspectrum of Predictor Hessian, to analyze distributed and federated learning frameworks.

| Analysis | Iteration Complexity | Assumption |
|---|---|---|
| This work (Theorem 4.1) | $\mathcal{O}\left(\left(\frac{\beta}{\mu} + \frac{\kappa\varepsilon}{\mu\sqrt{m}}\right)\log\frac{1}{\epsilon}\right)$ | PL, RSS |
| Song et al. [62] (Theorem E.1) | $\mathcal{O}\left(\left(\frac{\beta}{\mu} + \frac{p\beta\varepsilon}{\mu}\right)\log\frac{1}{\epsilon}\right)$ | SC, $\beta$-smoothness |

(a) K=1 local steps

| Analysis | Iteration Complexity | Assumption |
|---|---|---|
| This work (Theorem 4.2) | $\mathcal{O}\left(\left(\frac{\beta}{\mu^2\epsilon} + \frac{\kappa\varepsilon}{\sqrt{m}\mu^2\epsilon}\right)\log\frac{1}{\epsilon}\right)$ | PL, RSS, Bounded Gradient |
| Song et al. [62] (Theorem F.10 ($\sigma$=0)) | $\mathcal{O}\left(\frac{p\beta K}{\mu}\log\frac{1}{\epsilon}\right)$ | SC, $\beta$-smoothness |

(b) K>1 local steps

Table 1: Comparison of iteration complexities and assumptions with prior work. Our results scale with $\kappa$ which captures the intrinsic dimension [29] of the model rather than ambient dimension $p$. SC refers to strong-convexity and PL refers to PL-condition. $\mu$ refers to the PL constant as well as the strong convexity parameter under SC assumption. According to Lemma C.2 the loss function $\mathcal{L}$ can be shown to be $\beta$-smooth and $m$ refers to the width of the neural network.

## 2   Related Work

**Communication-Efficient Distributed Learning.**   The high cost of communicating model updates between the clients and the server has motivated a recent interest in improving communication efficiency in distributed and federated learning. One common strategy called `FedAvg` [50] enables less frequent communication by letting the client perform multiple local iterations at every round. Another common approach is to compress the model updates before the communication to reduce the cost of each round. These efforts can broadly be categorized into sparsification [3, 73, 44], quantization [64, 4, 74], low-rank factorization [67, 51, 71], sketching [59, 62, 34], and sparse subnetwork training techniques [31, 40, 41, 47, 32]. While some of these compression methods are already unbiased [4], many are biased and have to be combined with other mechanisms to reduce the bias for better convergence [44, 59]. Linearity is another desired feature, as it simplifies the implementation of distributed learning with security-promoting techniques like secure aggregation (in the compressed dimension). Among the general compression approaches mentioned, sketching stands out as a simple linear and unbiased operation, allowing for computations in the reduced dimension before desketching. We do not propose a new compression method in this work but instead provide a substantially improved convergence analysis for sketching-based frameworks that breaks the dimension dependence–which explains why sketching would not explode the converge error in large models.

**Sketching.**   Over many years, sketching has been a fundamental tool for many applications, even before the surge of deep learning in 2010s [18, 25, 36] for low-rank approximation [65], graph sparsification [1], and least squares regression [21]. More recently, sketching has also found use in distributed and federated learning frameworks to reduce the dimension of the model updates for communication efficiency [34, 33, 59, 62, 27]. The linearity of these sketching-based frameworks has led to their successful integration with secure aggregation and differential privacy as well [16, 66, 62, 17]. Despite the empirical success of these sketching-based applications in distributed and federated learning, the existing upper bounds on the convergence errors have a dependence on the ambient dimension of the model–which limits their scalability to larger models. In this work, we provide a tighter convergence analysis and get rid of the dimension dependence, suggesting the promise of sketching at scale. The closest to our work is by Rothchild et al. [59] who also provide a

dimension-independent convergence bound for their sketching algorithm, called `FetchSGD` (which is a combination of `Count-Sketch` projection, `Top-r` sparsification, and bias reduction with error feedback), under the assumption that the model updates have heavy-hitters. We note that both the heavy-hitter assumption and the `Top-r` step (and the error feedback coming along to minimize the bias) are necessary to get rid of the dimension dependence in their convergence analysis. We avoid both of these restrictions. Our key contributions over `FetchSGD` are: (1) We do not make the heavy-hitter assumption since it does not hold in general. (2) We do not require `Top-r` sparsification to break the dimension dependence – this way, we have an unbiased sketching mechanism without the need for error feedback, which would increase the memory cost and make the integration with differential privacy mechanisms complicated.

**Notation.** For a quantity $x$, we use the notation $x_t$ to refer to the global variable at round $t$ shared by all the clients and the server, and $x_{c,t}$ to refer to the local variable for client $c$ at round $t$. For a positive integer $n$, $[n] = \{1, \cdots, n\}$. We use $\mathbb{I}^{p \times p}$ as the $p \times p$ identity matrix. We use $\mathbb{E}[\cdot]$ for expectation. For a vector $\mathbf{x}$, we use $\|\mathbf{x}\|$ or $\|\mathbf{x}\|_2$ to denote its $\mathbb{L}_2$ norm. For a matrix $A$, we use $\|A\|$ or $\|A\|_2$ for the spectral norm of $A$. We use $\nabla_\theta \mathcal{L}(\theta^{'}) = \frac{\partial \mathcal{L}}{\partial \theta}|_{\theta = \theta'}$. For a random vector $\mathbf{y}$, $\|\mathbf{y}\|_{\psi_2}$ denotes its sub-Gaussian norm [70]. The notations $\tilde{\mathcal{O}}(t), \tilde{\Omega}(t), \tilde{\Theta}(t)$ are the same as the common $\mathcal{O}(t), \Omega(t), \Theta(t)$ but they hide the dependence on logarithmic terms. $\text{polylog}(n)$ denotes $\mathcal{O}(\log^k(n))$ for some $k$.

# 3 Sketching for Distributed Learning

We consider a distributed learning framework with $C$ clients, each client $c = 1, \ldots, C$ having a local dataset $\mathcal{D}_c = \{x_{i,c}, y_{i,c}\}_{i=1}^{n_c}$ of size $n_c$ and a local loss $\mathcal{L}_c : \mathbb{R}^p \to \mathbb{R}$ defined as $\mathcal{L}_c(\theta) = \frac{1}{n_c} \sum_{i=1}^{n_c} \ell(y_{i,c}, \hat{y}_{i,c})$, where $\theta \in \mathbb{R}^p$ is the parameter vector, $\hat{y}_{i,c} := f(\theta; \mathbf{x}_{i,c})$ is the prediction of the model for input $\mathbf{x}_{i,c}$, and $\ell : \mathbb{R} \times \mathbb{R} \to \mathbb{R}$ is a loss function that measures the error between $\hat{y}_{i,c}$ and the groundtruth $y_{i,c}$. Our goal is to minimize the empirical loss $\mathcal{L}(\theta)$:

$$\mathcal{L}(\theta) = \frac{1}{C} \sum_{c=1}^{C} \mathcal{L}_c(\theta). \tag{1}$$

Unlike much of the existing literature [59, 33, 62], our analysis does not ignore the fact that the predictor $f$ is a deep learning model – i.e., the loss $\mathcal{L}_c$ is not just an arbitrary smooth loss. The motivation behind this focus is that many of the modern models being used in distributed learning are indeed deep learning models. Interestingly, the losses associated with the deep learning models have a certain second-order structure beyond basic smoothness which will be the key to our sharper analysis. We also demonstrate that just assuming that the loss is smooth, as is typically done in most of the existing literature [59, 33, 62], does not avail one of the sharper analysis we introduce – making the dimension dependence unavoidable.

Following standard literature [22, 54, 45, 6], we consider a fully-connected feed-forward neural network $f$ of depth $L$, with widths $m$ and activations $\alpha^{(l)}$ for each layer $l \in [L] := \{1, \ldots, L\}$, described as:

$$\alpha^{(0)}(\mathbf{x}) = \mathbf{x},$$
$$\alpha^{(l)}(\mathbf{x}) = \phi\left(\frac{1}{\sqrt{m_{l-1}}} W_t^{(l)} \alpha^{(l-1)}(\mathbf{x})\right), \quad \forall l \in [L] \tag{2}$$
$$f(\theta; \mathbf{x}) = \alpha^{(L+1)}(\mathbf{x}) = \frac{1}{\sqrt{m_L}} \mathbf{v}_t^\top \alpha^{(L)}(\mathbf{x}),$$

where $W_t^{(l)} \in \mathbb{R}^{m_l \times m_{l-1}}$ is the layer-wise weight matrix for layer $l \in [L]$ and $\mathbf{v}_t \in \mathbb{R}^{m_L}$ is the last layer vector at iteration $t$, $\phi(\cdot)$ is the smooth (pointwise) activation function, and $m_0 = \dim(\mathbf{x}) = d$. We denote the total set of parameters as

$$\theta_t := ((\vec{W}_t^{(1)})^\top, \ldots, (\vec{W}_t^{(L)})^\top, \mathbf{v}_t^\top)^\top \in \mathbb{R}^p. \tag{3}$$

For simplicity, we will assume that the width of all the layers is the same, i.e., $m_l = m$ for all $l \in [L]$, and thus $p = (L-1)m^2 + md + m$. We consider deep models with only one output, i.e., $f(\theta; \mathbf{x}) \in \mathbb{R}$, but our results can be extended to multi-dimensional outputs.

As prevalent in literature [22, 6], we make the following assumptions regarding the activation function $\phi$, loss function $\ell$, and the initialization parameter $\theta_0$ which hold true for the commonly used activation functions, loss functions, and initializations used in practice.

**Assumption 3.1** (**Activation function**). *The activation $\phi$ is 1-Lipschitz and $\beta_\phi$-smooth, i.e. $|\phi'| \leq 1$ and $|\phi''| \leq \beta_\phi$.*

**Assumption 3.2** (**Initialization**). *We initialize $\theta_0$ with $w_{0,ij}^{(l)} \sim \mathcal{N}(0, \sigma_0^2)$ for $l \in [L]$ where $\sigma_0 = \frac{\sigma_1}{2\left(1 + \frac{\sqrt{\log m}}{\sqrt{2m}}\right)}, \sigma_1 > 0$, and $\mathbf{v}_0$ is a random unit vector with $\|\mathbf{v}_0\|_2 = 1$.*

**Assumption 3.3** (**Loss function**). *The loss $\ell_i = \ell(y_i, \hat{y}_i)$ with $\ell_i' = \frac{d\ell_i}{d\hat{y}_i}, \ell_i'' = \frac{d^2\ell_i}{d\hat{y}_i}$ is (i) Lipschitz, i.e., $|\ell_i'| \leq c_l$, and (ii) $\ell_i''$ and smooth $\ell_i'' \leq c_s$ for some $c_l, c_s > 0$.*

### 3.1 Sketching-Based Distributed Learning

Random sketching [75, 48] is a compression technique that uses random projections to reduce the dimensionality and helps speed up computations. These random linear mappings can be represented by sketching matrices $R \in \mathbb{R}^{b \times p}$ where typically $b \ll p$. Examples of sketching matrices include `Count-Sketch` [14], Subsampled Randomized Hadamard Transforms (`SRHT`) [13], and sparse Johnson–Lindenstrauss (`JL`) transforms [19]. In this work, we use sketching matrices to compress local updates before sending them to the server and refer to the operation of recovering true gradient vectors from the sketched updates as "desketching".

We outline the sketching-based distributed learning framework in Algorithm 1. Each client receives a random seed from the server to initialize the local parameters $\theta_{c,1}$, and generate a sketching matrix $R$. At each local step $k \in [1, \cdots, K]$, each client performs local gradient descent (GD) over their local dataset $D_c$. At each communication round, the client accumulates the changes over $K$-local steps, sketches the local updates, and sends the sketched update to the server. The server then aggregates the sketched changes and sends the aggregated sketched updates back to the clients. To update the local parameters, each client needs to recover an unbiased estimate of the true vector from the aggregated sketched update. We call this the desk (desketch) operation (Line 9), for which we use the transpose of the sketching matrix $R$. Each client then desketches the received aggregated sketched updates by applying desk and updates their local parameters. We refer to the sketching and desketching operations using the sk and desk operators defined as:

$$\mathrm{sk} := R \in \mathbb{R}^{b \times p} \quad \text{(Sketching)} , \tag{4}$$

$$\mathrm{desk} := R^\top \in \mathbb{R}^{p \times b} \quad \text{(Desketching)} . \tag{5}$$

While we use the same sketching matrix across communication rounds $t = 1, \ldots, T$, in general, using different matrices for each round does not affect the analysis.

**Choice of sketching matrix:** We use a $(1/\sqrt{b})$-sub-Gaussian matrix as the choice of sketching matrix. We say $R \in \mathbb{R}^{b \times p}$ is a $(1/\sqrt{b})$-sub-Gaussian matrix [69] if each row $R_i$ is an independent mean-zero, sub-Gaussian isotropic random-vector such that $\|R_i\|_{\psi_2} \leq 1/\sqrt{b}$. We assume $\mathbb{E}[R^\top R] = \mathbb{I}_{p \times p}$. From the above definition, we can see that for $g_1, g_2 \in \mathbb{R}^p$

$$R(g_1 + g_2) = Rg_1 + Rg_2 \qquad \textbf{(Linearity)},$$

$$\underset{\mathbb{R} \sim \Pi}{\mathbb{E}}[R^\top Rg] = g \qquad \textbf{(Unbiasedness)}. \tag{6}$$

### 3.2 Limitations of the Existing Analyses

When analyzing the convergence rates of the sketching-based distributed learning frameworks, previous works [62, 59] assume that the loss function $\mathcal{L}$ is $\beta$-smooth, i.e,

$$\mathcal{L}(\theta') \leq \mathcal{L}(\theta) + \langle \nabla\mathcal{L}(\theta), \theta' - \theta \rangle + \frac{\beta}{2}\|\theta' - \theta\|^2 . \tag{7}$$

In the model updates based on sketching followed by desketching, the term $\|\theta' - \theta\|^2$ effectively yields a term of the form $\|R^\top Rg\|_2^2$, where $g \in \mathbb{R}^p$ stands for a suitable gradient on the full model. While $\mathbb{E}[R^\top Rg] = g$, i.e., the desk-sk operation is unbiased, we unfortunately have $\|R^\top Rg\|_2^2 = \Theta(\frac{p}{b}\|g\|_2^2)$

**Algorithm 1** Sketching-Based Distributed Learning.

---

**Hyperparameters:** server learning rate $\eta_{\text{global}}$, local learning rate $\eta_{\text{local}}$.
**Inputs:** local datasets $\mathcal{D}_c$ of size $n_c$ for clients $c = 1, \ldots, C$, number of communication rounds $T$.
**Output:** final model $\theta_T$.

1:  Broadcast a random SEED to the clients.
2:  **for** $t = 1, \ldots, T$ **do**
3:      **On Client Nodes:**
4:      **for** $c = 1, \ldots, C$ **do**
5:          **if** $t = 1$ **then**
6:              Receive the random SEED from the server. Initialize the local model $\theta_{c,1} \in \mathbb{R}^p$ and generate the sketching matrix $R \in \mathbb{R}^{b \times p}$ (hence sk, desk) using the random SEED.
7:          **else**
8:              Receive $\text{sk}(\bar{\Delta}_{t-1})$ from the server.
9:              Desketch and update the model parameters $\theta_t \leftarrow \theta_{t-1} + \text{desk}(\text{sk}(\bar{\Delta}_{t-1}))$.
10:             Assign the local model's parameters $\theta_{c,t} \leftarrow \theta_t$ to be updated locally.
11:         **end if**
12:         **for** $k = 1, \ldots, K$ **do**
13:             $\theta_{c,t} \leftarrow \theta_{c,t} - \eta_{\text{local}} \cdot \nabla_\theta \mathcal{L}_c(\theta_{c,t})$
14:         **end for**
15:         $\Delta_{c,t} \leftarrow \theta_{c,t} - \theta_t$
16:         Send sketched updates $\text{sk}(\Delta_{c,t})$ to the server.
17:     **end for**
18:
19:     **On the Server Node:**
20:     Receive sketched updates $\text{sk}(\Delta_{c,t})$ from clients $c = 1, \ldots, C$.
21:     Aggregate: $\text{sk}(\bar{\Delta}_t) \leftarrow \eta_{\text{global}} \cdot \frac{1}{C} \sum_{c=1}^{C} \text{sk}(\Delta_{c,t})$
22:     Broadcast $\text{sk}(\bar{\Delta}_t)$ to the clients.
23: **end for**

---

with high probability (see Lemma B.2 in Appendix). Thus, such an analysis picks up a dimension dependence $\Theta(p)$ which can be neutralized only if the sketching dimension is $b = \Omega(p)$. However, such a high-dimensional projection will be ill-conceived as we will not get the benefits of the sketching projection. We note that prominent recent work has all hit this dimension dependence. For instance, Song et al. [62] have the dimension dependence in all their results including communication complexity, and Rothchild et al. [59] discuss the dimension dependence in Appendix B.3 – and get around the dependence by using Top-r components of the gradient vector, with an analysis having to rely on heavy-hitter assumptions. Interestingly, Rothchild et al. [59] noted that the sketching-based distributed deep learning approach seemed to work fine empirically without getting the adverse effect of dimension dependence despite what their theoretical results (based on smoothness) suggest. Our work sheds light on this discrepancy and proves (see Section 4) why the sketching-based distributed learning approach *in its simplest form* (see Algorithm 1) does not pick up the dimension dependence.

### 3.3 Restricted Strong Smoothness (RSS)

In this section, we describe the RSS property, an interesting property of deep learning losses that we use to derive dimension-independent convergence guarantees for sketch-DL. Using Taylor's expansion, the loss at $\theta = \theta'$ can be written as:

$$\mathcal{L}(\theta') = \mathcal{L}(\theta_t) + \langle \theta' - \theta_t, \nabla_\theta \mathcal{L}(\theta_t) \rangle + \frac{1}{2}(\theta' - \theta_t)^\top \nabla_\theta^2 \mathcal{L}(\tilde{\theta})(\theta' - \theta_t), \qquad (8)$$

where $\tilde{\theta} = \epsilon \theta' + (1 - \epsilon)\theta_t, \epsilon \in [0, 1]$ and $\nabla_\theta^2 \mathcal{L}(\tilde{\theta})$ is the Hessian of the loss. Several prior works [77, 61, 60] have studied the loss Hessian by decomposing the Hessian into the Gauss-Newton matrix (**G**) and averaged Hessian of the predictors as (**H**):

$$\nabla_\theta^2 \mathcal{L}(\theta) \coloneqq \mathbf{G} + \mathbf{H}, \qquad (9)$$

$$\mathbf{G} = \frac{1}{C} \sum_{c=1}^{C} \left( \frac{1}{n_c} \sum_{i=1}^{n_c} \ell_{i,c}'' \nabla f_{i,c} \nabla f_{i,c}^\top \right), \qquad (10)$$

$$\mathbf{H} = \frac{1}{C} \sum_{c=1}^{C} \left( \frac{1}{n_c} \sum_{i=1}^{n_c} \ell'_{i,c} \nabla^2 f_{i,c} \right) = \frac{1}{C} \left( \frac{1}{n_c} \sum_{i=1}^{n_c} H_{i,c} \right), \tag{11}$$

where $\nabla f_{i,c} = \nabla_\theta f(\theta; \mathbf{x}_{i,c})$ and $H_{i,c} = \ell'_{i,c} \nabla^2 f_{i,c} = \ell'_{i,c} \nabla^2_\theta f(\theta; \mathbf{x}_{i,c})$ respectively. Recent works [45, 6, 22] have derived sharp upper bounds on the spectral norm of the predictor Hessian: $\|\nabla^2 f\| = \Lambda_{max}(\nabla^2 f) = \mathcal{O}(1/\sqrt{m})$. This leads to the following restrcited smoothness property of deep learning losses:

$$\mathcal{L}(\theta') \leq \mathcal{L}(\theta_t) + \langle \theta' - \theta_t, \nabla_\theta \mathcal{L}(\theta_t) \rangle + \frac{1}{2C} \sum_{c=1}^{C} \frac{1}{n_c} \sum_{i=1}^{n_c} \ell''_{i,c} \langle \nabla f_{i,c}, \theta' - \theta_t \rangle^2 + \frac{c_0}{\sqrt{m}} \|\theta' - \theta_t\|^2.$$

In contrast to the $\beta$-smoothness assumption which is common in optimization literature, this new property shows that deep learning losses exhibit strong smoothness in a *restricted* set of directions. The effect of strong smoothness suitably manifests for $\theta'$ such that $|\cos(\theta' - \theta_t, \nabla f_{i,c})| \geq \kappa > 0$ for a restricted set of directions. In other directions, the strong smoothness constant is $\mathcal{O}(1/\sqrt{m})$, i.e., a tiny value. For our analysis, we use an even sharper perspective on RSS based on the eigenvalues of predictor Hessian, $H_{i,c} = \ell'_i \nabla^2 f_{i,c}$. While the smoothness perspective based on the spectral norm of $\mathcal{O}(1/\sqrt{m})$ is promising, it implicitly assumes all directions have this level of smoothness which would impact the analysis since there are $p = O(Lm^2)$ directions. Based on empirical evidence [55, 26], most directions have smoothness much smaller than $1/\sqrt{m}$, and an analysis based on the eigenvalues of the $H_{i,c}$ captures this sharper perspective as opposed to picking up a dependence on the ambient dimension $p$.

## 4 RSS-based analysis for Sketching-based Distributed Learning

In this section, we analyze the convergence of the sketching-based distributed learning approach, summarized in Algorithm 1, using the RSS property of deep learning losses and provide novel dimension-independent convergence results. First, we state our key assumptions in Section 4.1 and explain why they are well-supported by recent work. Then, we provide our novel analysis that eliminates the dimension dependence for the single-local step ($K = 1$) case in Section 4.2. Next in Section 4.3, we extend our analysis to the multiple-local step ($K > 1$) case. We conclude in Section 4.4 by deriving the communication cost of Algorithm 1 under our novel convergence analysis.

### 4.1 Assumptions

Before we introduce the assumptions, we would like to recall a key property.

**Definition 1** (PL condition). *Consider a loss function $\mathcal{L} : \mathbb{R}^p \to \mathbb{R}$ and the solution set $\mathcal{X}^* = \{\theta' : \theta' \in \operatorname*{argmin}_\theta \mathcal{L}(\theta)\}$ and we use $\mathcal{L}^*$ to denote the corresponding minimum value. Then, $\mathcal{L}$ is said to satisfy the Polyak-Łojasiewicz (PL) condition with constant $\mu$ if*

$$\frac{1}{2\mu} \|\nabla_\theta \mathcal{L}(\theta)\|_2^2 \geq \mathcal{L}(\theta) - \mathcal{L}^* .$$

**Remark 4.1.** PL condition [57] can be used to establish linear convergence of gradient descent while still being weaker than strong convexity. One can show that if a function $f$ is $\mu$-strongly convex then it also satisfies PL condition with the same constant $\mu$. Recent literature has shown that wide neural networks can be shown to satisfy some variant of PL condition [37, 46].

Relying on this recent evidence, we make the following assumption in our analysis:

**Assumption 4.1.** *Loss function $\mathcal{L}(\cdot)$ satisfies the PL condition with constant $\mu$.*

We further assume the following upper bound on the sum of eigenvalues of the $H_{i,c}$ as follows:

**Assumption 4.2** (Predictor Hessian Eigenspectrum). *Let $\Lambda_{i,c,t,1}, \Lambda_{i,c,t,2}, \cdots, \Lambda_{i,c,t,p}$ be the eigenvalues of the predictor Hessian $H_{i,c,t} = \ell'_{i,c} \nabla^2 f(\theta_t; \mathbf{x}_{i,c})$ for $t \in [T]$ and $\Lambda_{max} = \max_j |\Lambda_{i,c,t,j}|$ then there exists $\kappa = \mathcal{O}(1)$ such that $\sum_{j=1}^{p} |\Lambda_{i,c,t,j}| \leq \kappa \Lambda_{max}$.*

**Remark 4.2.** Several works [24, 49, 76] have shown that the spectrum of loss Hessian follows a bulk and outliers structure where the bulk can be attributed to **H** and follows power law trend [Figure 3(b), Figure 4 in Papyan [56]], motivating our assumption. Further empirical evidence is presented in Appendix G (refer to Figure 2), showing that for common networks, $\kappa$ is much smaller than $p$.

## 4.2 Single-Local Step ($K = 1$)

We now analyze the simpler setting where clients communicate their local updates after running a single local update step ($K = 1$). In this case, there is no local drift, i.e., all clients share the same local parameter vector. As a result, we can show that local update (Line 9 in Algorithm 1) can be written as $\theta_t \leftarrow \theta_{t-1} - \eta \operatorname{desk}(\operatorname{sk}(\nabla_\theta \mathcal{L}(\theta_{t-1})))$. We follow a similar analysis to Song et al. [62], but we exploit the second order structure of the deep learning losses, which helps avoid picking up dimension dependence due to the $\operatorname{desk}(\operatorname{sk}(\cdot))$ operator. We state the main theorem below with the full proof in Appendix D.1.

**Theorem 4.1** (Informal version of Theorem D.1). Set K=1 in Algorithm 1. Under Assumptions 4.1, and 4.2, for suitable constants $\varepsilon, \delta < 1$, learning rate $\eta = \eta_{\text{global}} \cdot \eta_{\text{local}}$ independent of $p$, and $b = \Omega\left(\frac{1}{\varepsilon^2} \operatorname{polylog}(\frac{p^2 NT}{\delta})\right)$, with probability at least $1 - \delta$, we have:

$$\mathcal{L}(\theta_T) - \mathcal{L}(\theta^*) \leq (\mathcal{L}(\theta_0) - \mathcal{L}(\theta^*)) e^{-\mu(1-\varepsilon)\eta T}, \tag{12}$$

where $\theta^*$ is a minimizer of (1).

In Theorem 4.1 (which also implies linear convergence), the sketching dimension, $b$, depends on $p$ only polylogarithmically, which explains the competitive performance of the sketching-based distributed learning frameworks with uncompressed GD without requiring any additional costly steps such as `Top-r` sparsification (as done by Rothchild et al. [59]), which also introduces a bias that needs to be suitably corrected.

## 4.3 Multiple-Local Step ($K > 1$)

Now, we provide the convergence result for Algorithm 1, for the more general case of multiple local steps ($K > 1$) at each round. Unlike the single-local step, here, the local clients' parameters drift during $K$ local steps and thus, we need to have additional assumptions to guarantee convergence. For the purpose of our analysis, we assume that the gradient norms are bounded, i.e., $\|\nabla_\theta \ell(\theta)\| \leq G$. We refer the reader to Theorem D.2 in the Appendix for a more formal statement. We now present an informal version of our theorem for the convergence of the multiple-local step case:

**Theorem 4.2** (Informal version of Theorem D.2). Let $\|\nabla_\theta \ell(\theta)\| \leq G$. Under Assumptions 4.1 and 4.2, for a suitable constants $\varrho, c_H$ and $\varepsilon < 1$, learning rate $\eta = \eta_{\text{global}} \eta_{\text{local}} < \frac{1}{2\mu K(1-\varepsilon)}$, and $b = \Omega\left(\frac{1}{\varepsilon^2} \operatorname{polylog}(\frac{TNp^2}{\delta})\right)$, with probability at least $1 - \delta$, we have:

$$\mathcal{L}(\theta_T) - \mathcal{L}(\theta^*) \leq (\mathcal{L}(\theta_0) - \mathcal{L}(\theta^*)) e^{-2(1-\varepsilon)\mu\eta KT} + \frac{\eta C_2(\varepsilon, m, \kappa) KG^2}{2\mu}, \tag{13}$$

where $\theta^*$ is a minimizer of the problem 1, $N$ is the number of training samples, and

$$C_2(\varepsilon, m, \kappa) = \mathcal{O}\left(\varrho^2 + \frac{c_H}{\sqrt{m}}\right) + \mathcal{O}\left(\frac{\varepsilon\kappa c_H}{\sqrt{m}}\right). \tag{14}$$

We refer the reader to Appendix D.2 for the full proof.

## 4.4 Communication Efficiency

As a direct consequence of our analysis, we now provide an improved communication complexity for Algorithm 1 compared to prior works [62] that build their analysis solely based on the smoothness assumption without taking advantage of the RSS property of the deep learning losses. With this, we manage to eliminate the linear dependence of the communication complexity on the ambient dimension $p$ – providing a substantial improvement over the analysis of Song et al. [62]. Unlike Rothchild et al. [59], we break this dependence without requiring a `Top-r` sparsification step or a heavy-hitter assumption. In the theorem below, we state the required number of communication bits to achieve an $\epsilon$-optimal solution based on our sharper convergence analysis in the previous sections.

**Theorem 4.3** (Informal version of Theorem E.1). Under Assumptions 4.1 and 4.2, Algorithm 1 obtains an optimal solution satisfying the error

$$\mathcal{L}(\theta_T) - \mathcal{L}(\theta^*) \leq \epsilon, \tag{15}$$

using $\tilde{\mathcal{O}}\left( C \max \left\{ 1, \frac{C_2(\varepsilon, m, \kappa)G^2}{2\mu^2(1-\varepsilon)\epsilon} \right\} \log \left( \frac{\mathcal{L}(\theta_0) - \mathcal{L}(\theta^*)}{\epsilon} \right) \right)$ bits of communication.

We provide the full proof in the Appendix E. We state the result by Song et al. [62] for comparison:

$$\mathcal{O}\left( \frac{\beta C}{\mu} \max \left\{ p, \sqrt{\frac{G^2}{\mu\epsilon}} \right\} \log \left( \frac{\beta \mathbb{E}[\|\theta_0 - \theta^*\|_2^2]}{\epsilon} \right) \right). \tag{16}$$

Song et al. [62] assumes that the loss function $\mathcal{L}$ is $\beta$-smooth and the expectation is over the randomness of sketching. Note that our bound depends on $\kappa$ instead of the ambient dimension $p$, presenting a sharper bound. This shows the efficacy of our approach in improving the overall communication complexity over prior works.

## 5 Experimental Results

In this section, we provide a comparison of the sketching approach in Algorithm 1 with other common approaches such as local `Top-r` [44] and `FetchSGD` [59]. We note that both Local `Top-r` (as outlined in Algorithm 2 in Appendix F) and `FetchSGD` are biased algorithms and they are typically used with error feedback mechanisms to correct the bias. As the sub-Gaussian sketching matrix in Algorithm 1, we use `Count-Sketch`. This means the only difference between `FetchSGD` without error feedback and `Count-Sketch` is the extra global `Top-r` step at the server. We conducted our experiments on NVIDIA Titan X GPUs on an internal cluster server, using 1 GPU per one run.

We train ResNet-18 [28] on CIFAR-10 dataset [38] that is i.i.d. distributed to 100 clients. Each client performs 5 local gradient descent iterations (i.e., using full-batch of size 500) at every round. Figure 1 shows that `Count-Sketch`-based distributed learning approach in Algorithm 1 performs competitively with `FetchSGD`. This result highlights the potential of sketching alone, without additional modifications as in `FetchSGD`, to maintain competitive accuracy. Additionally, the error-feedback free approach enables compatibility with Differential Privacy(DP) techniques which we leave as future work.

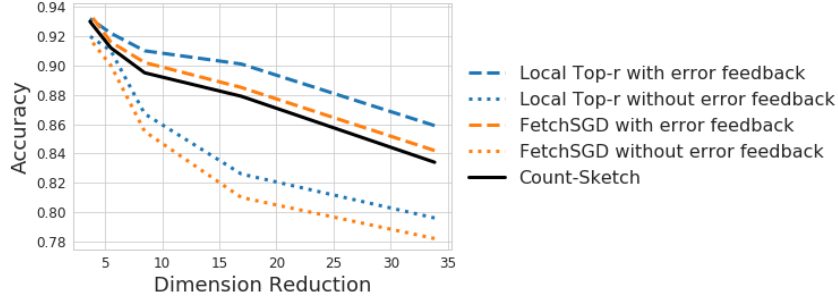

Figure 1: Communication Efficiency. `Count-Sketch` algorithm in Algorithm 1 against local `Top-r` [44] and `FetchSGD` [59], with and without error feedback. 100 clients run 5 local iterations with full-batch at every round.

We note that we do not claim novelty of any of the methods discussed in this section as they have been introduced [44, 34], improved [59], and analyzed [62] extensively in prior work. We present these empirical comparisons for completeness and to provide support for why a tight analysis of sketching is important, given the mismatch between its empirical success and the dimension dependence in existing convergence analysis as highlighted in Appendix B.3 of Rothchild et al. [59].

## 6 Discussion and Conclusion

We provide a significantly improved convergence analysis for sketching-based distributed learning frameworks by exploiting the properties of the deep learning losses, such as restricted strong smooth-

ness. This allows us to break the dimension dependence in the convergence error, and consequently, the communication cost – a milestone prior work could not achieve due to relying only on the smoothness assumption of the losses, i.e., ignoring the more special properties of "deep models." By breaking this dimension dependency in the convergence analysis and communication cost, we hope to motivate the use of sketching for larger models. One of the many exciting future extensions is to revisit the privacy analysis of private sketching mechanisms using our findings.

**Limitations.** We provide a sharper analysis for sketching in distributed learning. In future work, we plan to extend our analysis to federated learning by allowing client dropout.

**Broader Impact.** This paper provides a tighter analysis for sketching-based distributed learning and federated learning frameworks. We expect this work to be helpful for the community as it contributes to the efforts in making machine learning models more decentralized, accessible, and trustworthy.

## 7 Acknowledgements

The authors would like to thank the anonymous reviewers of NeurIPS for their valuable feedback and suggestions. AB acknowleges support in part by the National Science Foundation (NSF) through awards IIS 21-31335, OAC 21-30835, DBI 20-21898, as well as a C3.ai research award. BI was supported in part by a Google PhD Fellowship. SK acknowledges support by NSF III 2046795, IIS 1909577, CCF 1934986, NIH 1R01MH116226-01A, NIFA award 2020-67021-32799, the Alfred P. Sloan Foundation, and Google Inc.

## Footnotes

[2]We use `Top-r` instead of the more commonly used `Top-k` since we reserve the notation $k$ to refer to the local steps at clients.

## References

[1] K. J. Ahn, S. Guha, and A. McGregor. Graph sketches: sparsification, spanners, and subgraphs. In *Proceedings of the 31st ACM SIGMOD-SIGACT-SIGAI symposium on Principles of Database Systems*, pages 5–14, 2012.

[2] N. Ailon and B. Chazelle. Approximate nearest neighbors and the fast johnson-lindenstrauss transform. In *Proceedings of the thirty-eighth annual ACM symposium on Theory of computing*, pages 557–563, 2006.

[3] A. Aji and K. Heafield. Sparse communication for distributed gradient descent. In *EMNLP 2017: Conference on Empirical Methods in Natural Language Processing*. Association for Computational Linguistics (ACL), 2017.

[4] D. Alistarh, D. Grubic, J. Li, R. Tomioka, and M. Vojnovic. Qsgd: Communication-efficient sgd via gradient quantization and encoding. *Advances in neural information processing systems*, 30, 2017.

[5] A. Banerjee, S. Chen, F. Fazayeli, and V. Sivakumar. Estimation with norm regularization. *Advances in neural information processing systems*, 27, 2014.

[6] A. Banerjee, P. Cisneros-Velarde, L. Zhu, and M. Belkin. Restricted strong convexity of deep learning models with smooth activations. In *The Eleventh International Conference on Learning Representations*, 2023.

[7] L. P. Barnes, H. A. Inan, B. Isik, and A. Özgür. rtop-k: A statistical estimation approach to distributed sgd. *IEEE Journal on Selected Areas in Information Theory*, 1(3):897–907, 2020.

[8] R. B. Basat, S. Vargaftik, A. Portnoy, G. Einziger, Y. Ben-Itzhak, and M. Mitzenmacher. QUICK-FL: Quick unbiased compression for federated learning. *arXiv preprint arXiv:2205.13341*, 2022.

[9] J. H. Bell, K. A. Bonawitz, A. Gascón, T. Lepoint, and M. Raykova. Secure single-server aggregation with (poly) logarithmic overhead. In *Proceedings of the 2020 ACM SIGSAC Conference on Computer and Communications Security*, pages 1253–1269, 2020.

[10] J. Bernstein, Y.-X. Wang, K. Azizzadenesheli, and A. Anandkumar. signsgd: Compressed optimisation for non-convex problems. In *International Conference on Machine Learning*, pages 560–569. PMLR, 2018.

[11] R. Bommasani, D. A. Hudson, E. Adeli, R. Altman, S. Arora, S. von Arx, M. S. Bernstein, J. Bohg, A. Bosselut, E. Brunskill, et al. On the opportunities and risks of foundation models. *arXiv preprint arXiv:2108.07258*, 2021.

[12] K. Bonawitz, V. Ivanov, B. Kreuter, A. Marcedone, H. B. McMahan, S. Patel, D. Ramage, A. Segal, and K. Seth. Practical secure aggregation for federated learning on user-held data. *arXiv preprint arXiv:1611.04482*, 2016.

[13] C. Boutsidis and A. Gittens. Improved matrix algorithms via the subsampled randomized hadamard transform. *SIAM Journal on Matrix Analysis and Applications*, 34(3):1301–1340, 2013. doi: 10.1137/120874540. URL `https://doi.org/10.1137/120874540`.

[14] M. Charikar, K. Chen, and M. Farach-Colton. Finding frequent items in data streams. In *International Colloquium on Automata, Languages, and Programming*, pages 693–703. Springer, 2002.

[15] S. Chen and A. Banerjee. Structured estimation with atomic norms: General bounds and applications. *Advances in Neural Information Processing Systems*, 28, 2015.

[16] W.-N. Chen, C. A. C. Choo, P. Kairouz, and A. T. Suresh. The fundamental price of secure aggregation in differentially private federated learning. In *International Conference on Machine Learning*, pages 3056–3089. PMLR, 2022.

[17] W.-N. Chen, B. Isik, P. Kairouz, A. No, S. Oh, and Z. Xu. Improved communication-privacy trade-offs in $l\_2$ mean estimation under streaming differential privacy. In *Forty-first International Conference on Machine Learning*, 2024. URL `https://openreview.net/forum?id=x1G7ieRgRd`.

[18] G. Cormode and S. Muthukrishnan. An improved data stream summary: the count-min sketch and its applications. *Journal of Algorithms*, 55(1):58–75, 2005.

[19] A. Dasgupta, R. Kumar, and T. Sarlos. A sparse johnson: Lindenstrauss transform. In *Proceedings of the Forty-Second ACM Symposium on Theory of Computing*, STOC '10, page 341–350, New York, NY, USA, 2010. Association for Computing Machinery. ISBN 9781450300506. doi: 10.1145/1806689.1806737. URL `https://doi.org/10.1145/1806689.1806737`.

[20] J. Dean, G. Corrado, R. Monga, K. Chen, M. Devin, M. Mao, M. Ranzato, A. Senior, P. Tucker, K. Yang, et al. Large scale distributed deep networks. *Advances in neural information processing systems*, 25, 2012.

[21] E. Dobriban and S. Liu. Asymptotics for sketching in least squares regression. *Advances in Neural Information Processing Systems*, 32, 2019.

[22] S. Du, J. Lee, H. Li, L. Wang, and X. Zhai. Gradient descent finds global minima of deep neural networks. In *International conference on machine learning*, pages 1675–1685. PMLR, 2019.

[23] C. Dwork, A. Roth, et al. The algorithmic foundations of differential privacy. *Foundations and Trends® in Theoretical Computer Science*, 9(3–4):211–407, 2014.

[24] B. Ghorbani, S. Krishnan, and Y. Xiao. An investigation into neural net optimization via hessian eigenvalue density. In *International Conference on Machine Learning*, pages 2232–2241. PMLR, 2019.

[25] M. Greenwald and S. Khanna. Space-efficient online computation of quantile summaries. *ACM SIGMOD Record*, 30(2):58–66, 2001.

[26] G. Gur-Ari, D. A. Roberts, and E. Dyer. Gradient descent happens in a tiny subspace, 2018.

[27] F. Haddadpour, B. Karimi, P. Li, and X. Li. Fedsketch: Communication-efficient and private federated learning via sketching. *arXiv preprint arXiv:2008.04975*, 2020.

[28] K. He, X. Zhang, S. Ren, and J. Sun. Deep residual learning for image recognition. In *Proceedings of the IEEE conference on computer vision and pattern recognition*, pages 770–778, 2016.

[29] I. C. F. Ipsen and A. K. Saibaba. Stable rank and intrinsic dimension of real and complex matrices, 2024. URL https://arxiv.org/abs/2407.21594.

[30] B. Isik, T. Weissman, and A. No. An information-theoretic justification for model pruning. In *International Conference on Artificial Intelligence and Statistics*, pages 3821–3846. PMLR, 2022.

[31] B. Isik, F. Pase, D. Gunduz, T. Weissman, and Z. Michele. Sparse random networks for communication-efficient federated learning. In *The Eleventh International Conference on Learning Representations*, 2023. URL https://openreview.net/forum?id=k1FHgri5y3-.

[32] B. Isik, F. Pase, D. Gunduz, S. Koyejo, T. Weissman, and M. Zorzi. Adaptive compression in federated learning via side information. In *International Conference on Artificial Intelligence and Statistics*, pages 487–495. PMLR, 2024.

[33] N. Ivkin, D. Rothchild, E. Ullah, I. Stoica, R. Arora, et al. Communication-efficient distributed sgd with sketching. *Advances in Neural Information Processing Systems*, 32, 2019.

[34] J. Jiang, F. Fu, T. Yang, and B. Cui. Sketchml: Accelerating distributed machine learning with data sketches. In *Proceedings of the 2018 International Conference on Management of Data*, pages 1269–1284, 2018.

[35] P. Kairouz, H. B. McMahan, B. Avent, A. Bellet, M. Bennis, A. N. Bhagoji, K. Bonawitz, Z. Charles, G. Cormode, R. Cummings, et al. Advances and open problems in federated learning. *Foundations and Trends® in Machine Learning*, 14(1–2):1–210, 2021.

[36] D. M. Kane and J. Nelson. Sparser johnson-lindenstrauss transforms. *Journal of the ACM (JACM)*, 61(1):1–23, 2014.

[37] H. Karimi, J. Nutini, and M. Schmidt. Linear convergence of gradient and proximal-gradient methods under the polyak-Łojasiewicz condition, 2020.

[38] A. Krizhevsky, G. Hinton, et al. Learning multiple layers of features from tiny images.(2009), 2009.

[39] Y. LeCun, Y. Bengio, and G. Hinton. Deep learning. *nature*, 521(7553):436–444, 2015.

[40] A. Li, J. Sun, B. Wang, L. Duan, S. Li, Y. Chen, and H. Li. Lotteryfl: Personalized and communication-efficient federated learning with lottery ticket hypothesis on non-iid datasets. *arXiv preprint arXiv:2008.03371*, 2020.

[41] A. Li, J. Sun, X. Zeng, M. Zhang, H. Li, and Y. Chen. Fedmask: Joint computation and communication-efficient personalized federated learning via heterogeneous masking. In *Proceedings of the 19th ACM Conference on Embedded Networked Sensor Systems*, pages 42–55, 2021.

[42] M. Li, D. G. Andersen, J. W. Park, A. J. Smola, A. Ahmed, V. Josifovski, J. Long, E. J. Shekita, and B.-Y. Su. Scaling distributed machine learning with the parameter server. In *11th USENIX Symposium on operating systems design and implementation (OSDI 14)*, pages 583–598, 2014.

[43] M. Li, D. G. Andersen, A. J. Smola, and K. Yu. Communication efficient distributed machine learning with the parameter server. *Advances in Neural Information Processing Systems*, 27, 2014.

[44] Y. Lin, S. Han, H. Mao, Y. Wang, and B. Dally. Deep gradient compression: Reducing the communication bandwidth for distributed training. In *International Conference on Learning Representations*, 2018.

[45] C. Liu, L. Zhu, and M. Belkin. On the linearity of large non-linear models: when and why the tangent kernel is constant. In *Advances in Neural Information Processing Systems*, 2020.

[46] C. Liu, L. Zhu, and M. Belkin. On the linearity of large non-linear models: when and why the tangent kernel is constant. *Advances in Neural Information Processing Systems*, 33:15954–15964, 2020.

[47] Y. Liu, Y. Zhao, G. Zhou, and K. Xu. Fedprune: Personalized and communication-efficient federated learning on non-iid data. In *International Conference on Neural Information Processing*, pages 430–437. Springer, 2021.

[48] M. W. Mahoney. Randomized algorithms for matrices and data. *Foundations and Trends® in Machine Learning*, 3(2):123–224, 2011. ISSN 1935-8237. doi: 10.1561/2200000035. URL `http://dx.doi.org/10.1561/2200000035`.

[49] M. W. Mahoney et al. Randomized algorithms for matrices and data. *Foundations and Trends® in Machine Learning*, 3(2):123–224, 2011.

[50] H. B. McMahan, F. Yu, P. Richtarik, A. Suresh, and D. Bacon. Federated learning: Strategies for improving communication efficiency. In *Proceedings of the 29th Conference on Neural Information Processing Systems (NIPS), Barcelona, Spain*, pages 5–10, 2016.

[51] A. Mohtashami, M. Jaggi, and S. Stich. Masked training of neural networks with partial gradients. In *International Conference on Artificial Intelligence and Statistics*, pages 5876–5890. PMLR, 2022.

[52] S. Negahban and M. J. Wainwright. Restricted strong convexity and weighted matrix completion: Optimal bounds with noise. *The Journal of Machine Learning Research*, 13(1):1665–1697, 2012.

[53] S. Negahban, B. Yu, M. J. Wainwright, and P. Ravikumar. A unified framework for high-dimensional analysis of $m$-estimators with decomposable regularizers. *Advances in neural information processing systems*, 22, 2009.

[54] Q. Nguyen. On the proof of global convergence of gradient descent for deep relu networks with linear widths, 2021.

[55] V. Papyan. Measurements of three-level hierarchical structure in the outliers in the spectrum of deepnet hessians, 2019.

[56] V. Papyan. The full spectrum of deepnet hessians at scale: Dynamics with sgd training and sample size, 2019.

[57] B. Polyak. Gradient methods for the minimisation of functionals. *USSR Computational Mathematics and Mathematical Physics*, 3(4):864–878, 1963. ISSN 0041-5553. doi: https://doi.org/10.1016/0041-5553(63)90382-3. URL `https://www.sciencedirect.com/science/article/pii/0041555363903823`.

[58] D. Povey, X. Zhang, and S. Khudanpur. Parallel training of deep neural networks with natural gradient and parameter averaging. In *ICLR (Workshop)*, 2015.

[59] D. Rothchild, A. Panda, E. Ullah, N. Ivkin, I. Stoica, V. Braverman, J. Gonzalez, and R. Arora. Fetchsgd: Communication-efficient federated learning with sketching. In *International Conference on Machine Learning*, pages 8253–8265. PMLR, 2020.

[60] L. Sagun, L. Bottou, and Y. LeCun. Eigenvalues of the hessian in deep learning: Singularity and beyond. *arXiv preprint arXiv:1611.07476*, 2016.

[61] L. Sagun, U. Evci, V. U. Guney, Y. Dauphin, and L. Bottou. Empirical analysis of the hessian of over-parametrized neural networks, 2018.

[62] Z. Song, Y. Wang, Z. Yu, and L. Zhang. Sketching for first order method: Efficient algorithm for low-bandwidth channel and vulnerability. In *International Conference on Machine Learning*, pages 32365–32417. PMLR, 2023.

[63] S. U. Stich, J.-B. Cordonnier, and M. Jaggi. Sparsified sgd with memory. *Advances in Neural Information Processing Systems*, 31, 2018.

[64] A. T. Suresh, X. Y. Felix, S. Kumar, and H. B. McMahan. Distributed mean estimation with limited communication. In *International Conference on Machine Learning*, pages 3329–3337. PMLR, 2017.

[65] J. A. Tropp, A. Yurtsever, M. Udell, and V. Cevher. Practical sketching algorithms for low-rank matrix approximation. *SIAM Journal on Matrix Analysis and Applications*, 38(4):1454–1485, 2017.

[66] E. Ullah, C. A. Choquette-Choo, P. Kairouz, and S. Oh. Private federated learning with autotuned compression. In *International Conference on Machine Learning*, pages 34668–34708. PMLR, 2023.

[67] S. Vargaftik, R. Ben-Basat, A. Portnoy, G. Mendelson, Y. Ben-Itzhak, and M. Mitzenmacher. Drive: One-bit distributed mean estimation. *Advances in Neural Information Processing Systems*, 34:362–377, 2021.

[68] S. Vargaftik, R. B. Basat, A. Portnoy, G. Mendelson, Y. B. Itzhak, and M. Mitzenmacher. Eden: Communication-efficient and robust distributed mean estimation for federated learning. In *International Conference on Machine Learning*, pages 21984–22014. PMLR, 2022.

[69] R. Vershynin. *Introduction to the non-asymptotic analysis of random matrices*, page 210–268. Cambridge University Press, 2012.

[70] R. Vershynin. *High-Dimensional Probability: An Introduction with Applications in Data Science*. Cambridge Series in Statistical and Probabilistic Mathematics. Cambridge University Press, 2018.

[71] T. Vogels, S. P. Karimireddy, and M. Jaggi. Powersgd: Practical low-rank gradient compression for distributed optimization. *Advances in Neural Information Processing Systems*, 32, 2019.

[72] M. J. Wainwright. *High-dimensional statistics: A non-asymptotic viewpoint*, volume 48. Cambridge university press, 2019.

[73] H. Wang, S. Sievert, S. Liu, Z. Charles, D. Papailiopoulos, and S. Wright. Atomo: Communication-efficient learning via atomic sparsification. *Advances in neural information processing systems*, 31, 2018.

[74] W. Wen, C. Xu, F. Yan, C. Wu, Y. Wang, Y. Chen, and H. Li. Terngrad: Ternary gradients to reduce communication in distributed deep learning. *Advances in neural information processing systems*, 30, 2017.

[75] D. P. Woodruff. Computational advertising: Techniques for targeting relevant ads. *Foundations and Trends® in Theoretical Computer Science*, 10(1–2):1–157, 2014. ISSN 1551-3068. doi: 10.1561/0400000060. URL `http://dx.doi.org/10.1561/0400000060`.

[76] Z. Xie, Q.-Y. Tang, Y. Cai, M. Sun, and P. Li. On the power-law hessian spectrums in deep learning, 2022.

[77] C. Xing, D. Arpit, C. Tsirigotis, and Y. Bengio. A walk with sgd. *arXiv preprint arXiv:1802.08770*, 2018.

[78] X. Zhang, Z. Bu, Z. S. Wu, and M. Hong. Differentially private sgd without clipping bias: An error-feedback approach, 2024.

## A  Sketching Guarantee

**Lemma A.1.** *For a randomised matrix $R \in \mathbb{R}^{b \times p}$ with i.i.d. subgaussian rows $R_i$, i.e. $\|R_i\|_{\psi_2} \leq 1/\sqrt{b}$ and $\mathbb{E}[R_i R_i^T] = \mathbb{I}^{p \times p}$ and for $U, V \subseteq \mathbb{R}^p$, such that $b = \Omega(\frac{\log^3(p|U||V|/\delta)}{\varepsilon^2})$ with probability atleast $1 - \delta$, we have:*

$$\sup_{u \in U, v \in V} |\langle Ru, Rv \rangle - \langle u, v \rangle| \leq \varepsilon \|u\| \|v\| \tag{17}$$

*Proof.* We use the following lemma from Song et al. [62]:

**Lemma A.2** (Lemma D.24 from [62]). *Let $R \in \mathbb{R}^{b \times p}$ denote a random Gaussian matrix. Then for any fixed vector $h \in \mathbb{R}^p$ and any fixed vector $g \in \mathbb{R}^p$, the following property holds:*

$$\Pr_{R \sim \Pi} \left[ |(g^\top R^\top R h) - (g^\top h)| > \frac{\log^{1.5}(p/\delta)}{\sqrt{b}} \|g\|_2 \|h\|_2 \right] \leq \Theta(\delta).$$

Following a similar proof as Song et al. [62], we can get a sub-Gaussian version of this lemma:

**Lemma A.3.** *Let $R \in \mathbb{R}^{b \times p}$ be a random sub-Gaussian matrix, with $\psi_2$ norm of each entry bounded by $\frac{1}{\sqrt{b}}$. Then we have:*

$$Pr \left[ \max_{i \neq j} |\langle R_{*,i}, R_{*,j} \rangle| \geq \frac{c\sqrt{\log(n/\delta)}}{\sqrt{b}} \right] \leq \Theta(\delta)$$

*Proof.* Note for $i \neq j$, $R_{*,i}, R_{*,j}$ are two independent sub-Gaussian vectors. Let $z_k = R_{k,i} R_{k,j}$ and $z = \langle R_{*,i}, R_{*,j} \rangle$. Then according to the definition of sub-Gaussian random variables, $z_k \in \text{SE}\left(\frac{c_1^2}{b^2}, \frac{c_1}{b}\right)$ is a sub-exponential random variable with an absolute constant $c_1$. Thus, we have $z = \sum_{k=1}^{b} z_k \in \text{SE}\left(\frac{c_1^2}{b}, \frac{c_1}{b}\right)$, by sub-exponential concentration Lemma B.7 in Song et al. [62] we have:

$$\Pr[|z| \geq t] \leq 2 \exp(-c_2 b \min\{t^2, t\}).$$

Picking $t = c_3 \sqrt{\log(p^2/\delta)/b}$, we have:

$$\Pr \left[ |\langle R_{*,i}, R_{*,j} \rangle| \geq \frac{c\sqrt{\log(p/\delta)}}{\sqrt{b}} \right] \leq \delta/p^2.$$

Taking the union bound over all $(i, j) \in [p] \times [p]$ and $i \neq j$, we complete the proof. $\qquad \square$

Then following the same proof with the only difference to apply Lemma A.3 instead of Lemma D.18 in Song et al. [62], we can get the following sub-Gaussian version of Lemma D.24 in Song et al. [62].

**Lemma A.4.** *Let $R \in \mathbb{R}^{b \times p}$ denote a random sub-Gaussian matrix, with $\psi_2$ norm of each row bounded by $\frac{1}{\sqrt{b}}$. Then for any fixed vector $h \in \mathbb{R}^p$ and any fixed vector $g \in \mathbb{R}^p$, the following properties hold:*

$$\Pr_{R \sim \Pi} \left[ |(g^\top R^\top R h) - (g^\top h)| > \frac{c \log^{1.5}(p/\delta)}{\sqrt{b}} \|g\|_2 \|h\|_2 \right] \leq \Theta(\delta).$$

Based on Lemma A.4, taking union bound over $u \in U, v \in V$ and setting $b = \Omega\left(\frac{C \log^3(p|U||V|/\delta)}{\varepsilon^2}\right)$, we get the claim. $\qquad \square$

# B Dimension dependence

We make use of the following lemma from Banerjee et al. [5] to show dimensional dependence in $\|R^\top Rg\|_2^2$:

**Lemma B.1** (Theorem 5 in Banerjee et al. [5]). *Let $A \subseteq S^{p-1}$ and let $X \in \mathbb{R}^{n \times p}$ be a design matrix with independent isotropic sub-Gaussian rows, i.e., $\|X_i\|_{\psi_2} \leq \kappa$ and $E[X_i X_i^T] = \mathbb{I}_{p \times p}$. Then, for absolute constants $\eta, c > 0$, with probability at least $1 - 2\exp(-\eta w^2(A))$, we have:*

$$\sup_{u \in A} \left| \frac{1}{n} \|Xu\|^2 - 1 \right| \;=\; \sup_{u \in A} \left| \frac{1}{n} \sum_{i=1}^{n} \langle X_i, u \rangle^2 - 1 \right| \;\leq\; c \frac{w(A)}{\sqrt{n}}. \tag{18}$$

*where $w(A)$ denotes the Gaussian width of any set $A$ given by : $w(A) = \mathbb{E}_g \left[ \sup_{a \in A} \langle a, g \rangle \right]$, where $g$ is an isotropic Gaussian random vector, i.e., $g \sim \mathcal{N}(0, \mathbb{I}_{p \times p})$.*

**Remark B.1.** By choosing $X$ to be a $\frac{1}{\sqrt{n}}$ sub-Gaussian matrix, we get the form :

$$\sup_{u \in A} \left| \|Xu\|^2 - 1 \right| \;\leq\; c \frac{w(A)}{\sqrt{n}}.$$

which we will use next.

**Lemma B.2** (Dimension Dependence in sub-Gaussian sketching). *Fix a vector $g \in \mathbb{R}^p$. For a sketching matrix $R$ defined earlier, we have for some constants $\eta_1, \eta_2$ with probability at least $1 - 2\exp(-\eta_1 \epsilon^2 b) - 2\exp(-\eta_2 \epsilon^2 p)$,*

$$\|R^\top Rg\|_2^2 \geq (1-\epsilon)^2 \frac{p}{b} \|g\|_2^2. \tag{19}$$

*Proof.* Let $\mathcal{G} \subseteq \mathbb{R}^p$. Gaussian width of $\mathcal{G}$, $w(\mathcal{A}) = \mathcal{O}(\sqrt{\log|\mathcal{G}|})$. For $g \in \mathcal{G}$ using Lemma B.1,

$$\Pr\left[ \sup_{g \in \mathcal{G}} \left| \|Rg\|_2^2 - \|g\|^2 \right| \geq c_1 \frac{w(\mathcal{A})}{\sqrt{b}} \|g\|_2^2 \right] \leq 2\exp(-\eta w^2(\mathcal{G})) \tag{20}$$

Similarly, we can define $\mathcal{B} = \{h : h = Rg \; \forall g \in \mathcal{G}\}$, Gaussian width $w(\mathcal{B}) = \mathcal{O}(\|R\|\sqrt{\log|\mathcal{G}|})$. Now setting the design matrix $X$ for Lemma B.1 to be $\sqrt{b/p} \cdot R^\top$:

$$\Pr\left[ \sup_{g \in G} \left| \frac{b}{p} \|R^\top Rg\|_2^2 - \|Rg\|^2 \right| \geq c_2 \frac{w(\mathcal{B})}{\sqrt{p}} \|Rg\|^2 \right] \leq 2\exp(-\eta w^2(\mathcal{B})) \tag{21}$$

Taking $\epsilon = \max\left\{ c_1 \frac{w(A)}{\sqrt{b}}, c_2 \frac{w(B)}{\sqrt{p}} \right\}$,

$$\Pr\left[ \sup_{g \in G} \left| \|Rg\|_2^2 - \|g\|_2^2 \right| \geq \epsilon \|g\|_2^2 \right] \leq 2\exp(-\eta_1 \epsilon^2 b)) \tag{22}$$

$$\text{and, } \Pr\left[ \sup_{g \in G} \left| \frac{b}{p} \|R^\top Rg\|_2^2 - \|Rg\|^2 \right| \geq \epsilon \|Rg\|^2 \right] \leq 2\exp(-\eta \epsilon^2 p) \tag{23}$$

Succintly, we can write with probability $1 - 2\exp(-\eta_1 \epsilon^2 b) - 2\exp(-\eta_2 \epsilon^2 p)$,

$$\|R^\top Rg\|_2^2 \geq (1-\epsilon)^2 \frac{p}{b} \|g\|_2^2 \tag{24}$$

$\qquad \square$

# C Background

We state some useful results from Banerjee et al. [6]:

**Lemma C.1** (Predictor Hessian Spectral Norm Bound, Theorem 4.1 and Lemma 4.1 in Banerjee et al. [6] ). *Under Assumptions 3.1 and 3.2, for any* $\mathbf{x} \in \mathcal{X}$, $\theta \in B_{\rho,\rho_1}^{\text{Spec}}(\theta_0)$, *with probability at least* $(1 - \frac{2(L+1)}{m})$, *we have*

$$\|\nabla_\theta^2 f(\theta; \mathbf{x})\|_2 \leq \frac{c_H}{\sqrt{m}} \quad and \quad \|\nabla_\theta f(\theta; \mathbf{x})\|_2 \leq \varrho, \tag{25}$$

*where,*

$$c_H = L(L^2\gamma^{2L} + L\gamma^L + 1) \cdot (1 + \rho_1) \cdot \psi_H \cdot \max_{l \in [L]} \gamma^{L-l} + L\gamma^L \max_{l \in [L]} h(l),$$

$$\gamma = \sigma_1 + \frac{\rho}{\sqrt{m}}, \quad h(l) = \gamma^{l-1} + |\phi(0)| \sum_{i=1}^{l-1} \gamma^{i-1},$$

$$\psi_H = \max_{1 \leq l_1 < l_2 \leq L} \left\{ \beta_\phi h(l_1)^2, \ h(l_1)\left(\frac{\beta_\phi}{2}(\gamma^2 + h(l_2)^2) + 1\right), \ \beta_\phi\gamma^2 h(l_1)h(l_2) \right\},$$

$$\varrho^2 = (h(L+1))^2 + \frac{1}{m}(1 + \rho_1)^2 \sum_{l=1}^{L+1} (h(l))^2 \gamma^{2(L-l)}.$$

**Lemma C.2** (**Local Smoothness**, Theorem 5.2 in Banerjee et al. [6]). *Under Assumptions 3.2 and 3.3, with probability at least* $(1 - \frac{2(L+1)}{m})$, $\forall \theta, \theta' \in B_{\rho,\rho_1}^{\text{Frob}}(\theta_0)$,

$$\ell(\theta') \leq \ell(\theta) + \langle \theta' - \theta, \nabla_\theta \mathcal{L}(\theta) \rangle + \frac{\beta}{2}\|\theta' - \theta\|_2^2, \quad with \quad \beta = c_s\varrho^2 + \frac{c_H c_l}{\sqrt{m}}, \tag{26}$$

## D  Analysis of Convergence

We will bound $T_1, T_2$ and $T_3$, which we introduce below:

$$\mathcal{L}(\theta) = \frac{1}{C}\sum_{c=1}^{C} \mathcal{L}_c(\theta) = \frac{1}{C}\sum_{c=1}^{C}\left(\frac{1}{n_c}\sum_{i=1}^{n_c} \ell(y_{i,c}, f(\theta; \mathbf{x}_{i,c}))\right) \tag{27}$$

$$\nabla_\theta \mathcal{L}(\theta) = \frac{1}{C}\sum_{c=1}^{C} \nabla_\theta \mathcal{L}_c(\theta) = \frac{1}{C}\left(\frac{1}{n_c}\sum_{i=1}^{n_c} \ell'_{i,c}\nabla_\theta f(\theta; \mathbf{x}_{i,c})\right) \tag{28}$$

$$\nabla_\theta^2 \mathcal{L}(\theta) = \frac{1}{C}\sum_{c=1}^{C}\left(\frac{1}{n_c}\left(\sum_{i=1}^{n_c} \ell''_{i,c}\nabla_\theta f(\theta; \mathbf{x}_{i,c})\nabla_\theta f(\theta; \mathbf{x}_{i,c})^\top + \sum_{i=1}^{n_c} \underbrace{\ell'_{i,c}\nabla_\theta^2 f(\theta; \mathbf{x}_{i,c})}_{H_{i,c}(\theta)}\right)\right) \tag{29}$$

where $\ell'_i, \ell''_i$ are the first and second order derivatives of the point-wise losses. For an iterate $\theta_{t-1}$, by the second-order Taylor expansion of the empirical loss around $\theta_{t-1}$ we have (ignoring h.o.t.),

$$\mathcal{L}(\theta_t) = \mathcal{L}(\theta_{t-1}) + \langle \nabla_\theta \mathcal{L}(\theta_{t-1}), \theta_t - \theta_{t-1}\rangle + \frac{1}{2}(\theta_t - \theta_{t-1})^\top \nabla_\theta^2 \mathcal{L}(\theta_{t-1})(\theta_t - \theta_{t-1}) \tag{30}$$

Using 29,

$$\mathcal{L}(\theta_t) = \mathcal{L}(\theta_{t-1}) + \underbrace{\langle \nabla_\theta \mathcal{L}(\theta_{t-1}), \theta_t - \theta_{t-1}\rangle}_{T_1} + \underbrace{\frac{1}{2C}\sum_{c=1}^{C}\left(\frac{1}{n_c}\sum_{i=1}^{n_c} \ell''_{i,c}\langle \nabla_\theta f(\theta_{t-1}; \mathbf{x}_{i,c}), \theta_t - \theta_{t-1}\rangle^2\right)}_{T_2}$$

$$+ \underbrace{\frac{1}{2C}\sum_{c=1}^{C}\left(\frac{1}{n_c}\sum_{i=1}^{n_c}(\theta_t - \theta_{t-1})^\top H_{i,c}(\theta_t - \theta_{t-1})\right)}_{T_3} \tag{31}$$

where, $H_{i,c} = H_{i,c}(\theta_{t-1})$ for brevity.

## D.1 Single-step scheme (K=1)

**Theorem D.1.** Under assumptions 4.1 and 4.2, set $K = 1$ in Algorithm 1. For a suitable constant $\varepsilon < 1$, $\eta = \eta_{\text{global}}\eta_{\text{local}} = \dfrac{2(1-\varepsilon)}{((1+\varepsilon)^2 c_s \varrho^2 + {}^{c_H c_l}/\sqrt{m}(1 + \kappa(2\varepsilon + \varepsilon^2)))}$ and $b = \Omega\left(\frac{1}{\varepsilon^2}\log^3(pT/\delta)\right)$ with probability at least $1 - \delta$, we have:

$$\mathcal{L}(\theta_T) - \mathcal{L}(\theta^*) \le \left(\mathcal{L}(\theta_0) - \mathcal{L}(\theta^*)\right) e^{-\mu(1-\varepsilon)\eta T} \tag{32}$$

where $\theta^*$ is the minimizer of the problem 1.

*Proof.*

$$\theta_t - \theta_{t-1} = -\eta_{\text{global}} \operatorname{desk}\left(\frac{1}{C}\sum_{c=1}^{C} \operatorname{sk}\left(\eta_{\text{local}}\nabla_\theta \mathcal{L}_c(\theta_{t-1})\right)\right) = -\eta \operatorname{desk} \operatorname{sk} \nabla_\theta \mathcal{L}(\theta_{t-1}) \tag{33}$$

where $\eta = \eta_{\text{global}}\eta_{\text{local}}$ for notation simplicity.

### D.1.1 Bounding $T_1$

$$\langle \theta_t - \theta_{t-1}, \nabla_\theta \mathcal{L}(\theta_{t-1})\rangle = -\eta \langle \operatorname{desk} \operatorname{sk} \nabla_\theta \mathcal{L}(\theta_{t-1}), \nabla_\theta \mathcal{L}(\theta_{t-1})\rangle \tag{34}$$

$$= -\eta \langle R\nabla_\theta \mathcal{L}(\theta_{t-1}), R\nabla_\theta \mathcal{L}(\theta_{t-1})\rangle \tag{35}$$

$$\overset{a}{\le} -\eta\|\nabla_\theta \mathcal{L}(\theta_{t-1})\|_2^2 + \eta\varepsilon_1\|\nabla_\theta \mathcal{L}(\theta_{t-1})\|_2^2 \tag{36}$$

where, $a$ follows from Lemma A.1 for some for suitable choice of $\varepsilon_1$.

### D.1.2 Bounding $T_2$

$$\sum_{i=1}^{n_c} \ell_{i,c}''\langle \theta_t - \theta_{t-1}, \nabla_\theta f(\theta_{t-1}; \mathbf{x}_{i,c})\rangle^2 \tag{37}$$

$$\le \eta^2 \sum_{i=1}^{n_c} \ell_{i,c}'' \left[\langle \nabla_\theta \mathcal{L}(\theta_{t-1}), \nabla_\theta f(\theta_{t-1}; \mathbf{x}_{i,c})\rangle^2 + \varepsilon_2^2 \varrho^2\|\nabla_\theta \mathcal{L}(\theta_{t-1})\|_2^2 + 2\langle \nabla_\theta \mathcal{L}(\theta_{t-1}), \nabla_\theta f(\theta_{t-1}; \mathbf{x}_{i,c})\rangle \varepsilon_2 \varrho\|\nabla_\theta \mathcal{L}(\theta_{t-1})\|_2\right] \tag{38}$$

$$\le \eta^2 \sum_{i=1}^{n_c} \ell_{i,c}'' \left[\langle \nabla_\theta \mathcal{L}(\theta_{t-1}), \nabla_\theta f(\theta_{t-1}; \mathbf{x}_{i,c})\rangle^2 + \varrho^2\varepsilon_2^2\|\nabla_\theta \mathcal{L}(\theta_{t-1})\|_2^2 + 2\varrho^2\varepsilon_2\|\nabla_\theta \mathcal{L}(\theta_{t-1})\|_2^2\right] \tag{39}$$

$$\le \eta^2 \sum_{i=1}^{n_c} \ell_{i,c}'' \left[\langle \nabla_\theta \mathcal{L}(\theta_{t-1}), \nabla_\theta f(\theta_{t-1}; \mathbf{x}_{i,c})\rangle^2 + \varrho^2\left(\varepsilon_2^2 + 2\varepsilon_2\right)\|\nabla_\theta \mathcal{L}(\theta_{t-1})\|_2^2\right] \tag{40}$$

$$\le \eta^2 \sum_{i=1}^{n_c} \ell_{i,c}''\langle \nabla_\theta \mathcal{L}(\theta_{t-1}), \nabla_\theta f(\theta_{t-1}; \mathbf{x}_{i,c})\rangle^2 + \eta^2\varrho^2(\sum_{i=1}^{n_c} \ell_{i,c}'')\left(\varepsilon_2^2 + 2\varepsilon_2\right)\|\nabla_\theta \mathcal{L}(\theta_{t-1})\|_2^2 \tag{41}$$

$$\le \eta^2 \sum_{i=1}^{n_c} \ell_{i,c}''(1 + \varepsilon_2)^2\varrho^2\|\nabla_\theta \mathcal{L}(\theta_{t-1})\|^2 \tag{42}$$

Thus,

$$\frac{1}{2C}\sum_{c=1}^{C}\left(\frac{1}{n_c}\sum_{i=1}^{n_c} \ell_{i,c}''\langle \nabla_\theta f(\theta_{t-1}; \mathbf{x}_{i,c}), \theta_t - \theta_{t-1}\rangle^2\right) \le \frac{\eta^2}{2}\left(1 + \varepsilon_2\right)^2 c_s \varrho^2\|\nabla_\theta \mathcal{L}(\theta_{t-1})\|_2^2 \tag{43}$$

for some for suitable choice of $\varepsilon_2$.

### D.1.3 Bounding $T_3$

Using eigen-decomposition of $H_{i,c}$,

$$(\theta_t - \theta_{t-1})^\top H_{i,c}(\theta_t - \theta_{t-1}) = \sum_{j=1}^{p} \Lambda_{j,i,c}\langle \theta_t - \theta_{t-1}, \mathbf{v}_{j,i,c}\rangle^2 \tag{44}$$

where $\Lambda_{j,i,c}, \mathbf{v}_{j,i,c}$ for $j \in [1, \cdots, p]$ are the eigen-values and eigen-vectors of $H_{i,c}$ respectively. Now,

$$\langle \theta_t - \theta_{t-1}, \mathbf{v}_{j,i,c}\rangle = -\eta\langle \text{desk sk } \nabla_\theta \mathcal{L}(\theta_{t-1}), \mathbf{v}_{j,i,c}\rangle \tag{45}$$

$$\leq -\eta\langle \nabla_\theta \mathcal{L}(\theta_{t-1}), \mathbf{v}_{j,i,c}\rangle + \eta\varepsilon_3 \|\nabla_\theta \mathcal{L}(\theta_{t-1})\|_2 \tag{46}$$

$$\sum_{j=1}^{p} \Lambda_{j,i,c}\langle \theta_t - \theta_{t-1}, \mathbf{v}_{j,i,c}\rangle^2 \tag{47}$$

$$\leq \sum_{j=1}^{p} \Lambda_{j,i,c}\left\{ \langle \nabla_\theta \mathcal{L}(\theta_{t-1}), \mathbf{v}_{j,i,c}\rangle^2 + 2\varepsilon_3 \|\nabla_\theta \mathcal{L}(\theta_{t-1})\|_2 \langle \nabla_\theta \mathcal{L}(\theta_{t-1}), \mathbf{v}_{j,i,c}\rangle + \varepsilon_3^2 \|\nabla_\theta \mathcal{L}(\theta_{t-1})\|_2^2 \right\} \tag{48}$$

$$\leq \sum_{j=1}^{p} \Lambda_{j,i,c}\left\{ \langle \nabla_\theta \mathcal{L}(\theta_{t-1}), \mathbf{v}_{j,i,c}\rangle^2 + (\varepsilon_3^2 + 2\varepsilon_3)\|\nabla_\theta \mathcal{L}(\theta_{t-1})\|_2^2 \right\} \tag{49}$$

$$\leq \sum_{j=1}^{p} \Lambda_{j,i,c}\langle \nabla_\theta \mathcal{L}(\theta_{t-1}), \mathbf{v}_{j,i,c}\rangle^2 + \left( \sum_{j=1}^{p} |\Lambda_{j,i,c}| \right)(\varepsilon_3^2 + 2\varepsilon_3)\|\nabla_\theta \mathcal{L}(\theta_{t-1})\|_2^2 \tag{50}$$

$$\leq \Lambda_{\max,i,c} \sum_{j=1}^{p} \langle \nabla_\theta \mathcal{L}(\theta_{t-1}), \mathbf{v}_{j,i,c}\rangle\rangle^2 + \left( \sum_{j=1}^{p} |\Lambda_{j,i,c}| \right)(\varepsilon_3^2 + 2\varepsilon_3)\|\nabla_\theta \mathcal{L}(\theta_{t-1})\|_2^2 \tag{51}$$

$$\leq \frac{c_l c_H}{\sqrt{m}}\|\nabla_\theta \mathcal{L}(\theta_{t-1})\|^2 + \frac{\kappa c_l c_H}{\sqrt{m}}(\varepsilon_3^2 + 2\varepsilon_3)\|\nabla_\theta \mathcal{L}(\theta_{t-1})\|^2 \tag{52}$$

where, $\Lambda_{\max,i,c} = \|H_{i,c}\|_2 = |\ell'_{i,c}| \cdot \|\nabla^2 f\|_2 = \frac{c_l c_H}{\sqrt{m}}$.

Thus,

$$\frac{1}{2C}\sum_{c=1}^{C}\left( \frac{1}{n_c}\sum_{i=1}^{n_c}(\theta_t - \theta_{t-1})^\top H_{i,c}(\theta_t - \theta_{t-1}) \right) \leq \frac{\eta^2 c_H c_l}{2\sqrt{m}}(1 + \kappa(2\varepsilon_3 + \varepsilon_3^2))\|\nabla_\theta \mathcal{L}(\theta_{t-1})\|_2^2 \tag{53}$$

for some for suitable choice of $\varepsilon_3$.

### D.1.4 Combining $T_1, T_2, T_3$

Combining $T_1, T_2, T_3$, we get:

$$\mathcal{L}(\theta_t) \leq \mathcal{L}(\theta_{t-1}) - \eta\|\nabla_\theta \mathcal{L}(\theta_{t-1})\|_2^2 + \eta\varepsilon_1\|\nabla_\theta \mathcal{L}(\theta_{t-1})\|_2^2 \tag{54}$$

$$+ \frac{\eta^2}{2}(1 + \varepsilon_2)^2 c_s \varrho^2 \|\nabla_\theta \mathcal{L}(\theta_{t-1})\|_2^2 + \frac{\eta^2 c_H c_l}{2\sqrt{m}}(1 + \kappa(2\varepsilon_3 + \varepsilon_3^2))\|\nabla_\theta \mathcal{L}(\theta_{t-1})\|_2^2 \tag{55}$$

$$\leq \mathcal{L}(\theta_{t-1}) - \left[ \eta - \eta\varepsilon_1 - \frac{\eta^2}{2}(1 + \varepsilon_2)^2 c_s \varrho^2 - \frac{\eta^2 c_H c_l}{2\sqrt{m}}(1 + \kappa(2\varepsilon_3 + \varepsilon_3^2)) \right]\|\nabla_\theta \mathcal{L}(\theta_{t-1})\|_2^2 \tag{56}$$

Using PL condition,

$$\mathcal{L}(\theta_t) \leq \mathcal{L}(\theta_{t-1}) \tag{57}$$

$$- \left[ \eta - \eta\varepsilon_1 - \frac{\eta^2}{2}(1+\varepsilon_2)^2 c_s\varrho^2 - \frac{\eta^2 c_H c_l}{2\sqrt{m}}(1+\kappa(2\varepsilon_3+\varepsilon_3^2)) \right] 2\mu\left(\mathcal{L}(\theta_{t-1}) - \mathcal{L}(\theta^*)\right) \tag{58}$$

$$\mathcal{L}(\theta_t) - \mathcal{L}(\theta^*) \leq (1-\beta)\left(\mathcal{L}(\theta_{t-1}) - \mathcal{L}(\theta^*)\right) \tag{59}$$

where,

$$\beta = 2\mu\eta\left(1 - \varepsilon_1 - \frac{\eta}{2}\left((1+\varepsilon_2)^2 c_s\varrho^2 + \frac{c_H c_l}{\sqrt{m}}(1+\kappa(2\varepsilon_3+\varepsilon_3^2))\right)\right) \tag{60}$$

Setting $\eta < \dfrac{2(1-\varepsilon_1)}{(1+\varepsilon_2)^2 c_s\varrho^2 + \frac{c_H c_l}{\sqrt{m}}(1+\kappa(2\varepsilon_3+\varepsilon_3^2))}$ gives a contraction map. If we iterate this recursion, we get:

$$\mathcal{L}(\theta_t) - \mathcal{L}(\theta^*) \leq (1-\beta)^t\left(\mathcal{L}(\theta_0) - \mathcal{L}(\theta^*)\right) \tag{61}$$

Set

$$\varepsilon_1 = \varepsilon_2 = \varepsilon_3 = \varepsilon \ , \ \eta = \frac{2(1-\varepsilon)}{((1+\varepsilon)^2 c_s\varrho^2 + c_H c_l/\sqrt{m}(1+\kappa(2\varepsilon+\varepsilon^2)))} \tag{62}$$

we get

$$\mathcal{L}(\theta_T) - \mathcal{L}(\theta^*) \leq (1 - \mu(1-\varepsilon)\eta)^T\left(\mathcal{L}(\theta_0) - \mathcal{L}(\theta^*)\right) \tag{63}$$

$$\leq \left(\mathcal{L}(\theta_0) - \mathcal{L}(\theta^*)\right)e^{-\mu(1-\varepsilon)\eta T} \tag{64}$$

$$\square$$

### D.1.5 Choice of sketching dimension $b$:

Using A.1, for each step $t \in [0, \cdots T]$, union bounding over $T_1, T_2$ and $T_3$ and over all T time-steps, Union bounding over all T time-steps, we get

$$b = \Omega\left(\frac{1}{\varepsilon^2}\log^3(p^2 NT/\delta)\right) \tag{65}$$

where $N = \sum_{i=1}^{C} n_c$ is the number of training samples.

### D.2 Multi-step scheme

**Theorem D.2.** Let $\|\nabla_\theta \ell(\theta)\| \leq G$. Under Assumption 4.1, for a suitable constant $\varepsilon < 1$, $\eta = \eta_{\text{global}}\eta_{\text{local}} < \frac{1}{2\mu K(1-\varepsilon)}$ and $b = \Omega\left(\frac{1}{\varepsilon^2}\log^3(p^2 NT|C||K|/\delta)\right)$, with probability at least $1-\delta$, we have:

$$\mathcal{L}(\theta_T) - \mathcal{L}(\theta^*) \leq \left(\mathcal{L}(\theta_0) - \mathcal{L}(\theta^*)\right)e^{-2(1-\varepsilon)\mu\eta Kt} + \frac{\eta C_2(\varepsilon, m, \kappa)KG^2}{2\mu(1-\varepsilon)} \tag{66}$$

where $\theta^*$ is the minimizer of the problem 1. And,

$$C_2(\varepsilon, m, \kappa) = \frac{1}{1-\varepsilon}\left[\left(c_s\varrho^2 + \frac{c_H c_l}{\sqrt{m}}\right)(1+\varepsilon) + \frac{1}{2}c_s(1+\varepsilon)^2\varrho^2 + \frac{c_l c_H}{2\sqrt{m}} + \frac{1}{2}(2\varepsilon+\varepsilon^2)\frac{c_l\kappa\mathbf{c}_H}{\sqrt{m}}\right] \tag{67}$$

where, $\varrho$ and $c_H$ are defined in Lemma C.1.

*Proof.* We assume that the gradients at each time step t are bounded i.e $\|\nabla_\theta \mathcal{L}(\theta)\|_2 \le G$ for some suitable constant G.

First, we introduce some notation: In the K-step case since the parameters are shared only in the sync step, we define $\theta_{c,t,k} \in \mathbb{R}^p$ as the local parameters for the client $c$ at time step $t$ and step $k \in [1, 2, \cdots, K]$. Based on the sync step in Alg: 1, we can see that $\theta_{c,t,0}$ is the same for all clients $c \in C$ and thus, we denote it as $\theta_t$, i,e. $\theta_{c,t,0} = \theta_t \forall c \in C$. Thus, we can write the update in the sync step as:

$$\theta_t - \theta_{t-1} = -\eta_{\text{global}} \text{ desk} \left( \frac{1}{C} \sum_{c=1}^{C} \text{sk} \left( \eta_{\text{local}} \sum_{k=0}^{K-1} \nabla_\theta \mathcal{L}_c(\theta_{c,t-1,k}) \right) \right) \tag{68}$$

$$= -\eta_{\text{global}} \eta_{\text{local}} \left( \frac{1}{C} \sum_{c=1}^{C} \left( \sum_{k=0}^{K-1} \text{desk sk } \nabla_\theta \mathcal{L}_c(\theta_{c,t-1,k}) \right) \right) \tag{69}$$

$$= -\frac{\eta}{C} \left( \sum_{c=1}^{C} \left( \sum_{k=0}^{K-1} \text{desk sk } \nabla_\theta \mathcal{L}_c(\theta_{c,t-1,k}) \right) \right) \tag{70}$$

where,

$$\eta = \eta_{\text{global}} \eta_{\text{local}}.$$

Similar to Section D.1, we can bound $T_1, T_2$ and $T_3$ in 31 as:

### D.2.1   Bounding $T_1$

$$\langle \nabla_\theta \mathcal{L}(\theta_{t-1}), \theta_t - \theta_{t-1} \rangle \tag{71}$$

$$= \langle \nabla_\theta \mathcal{L}(\theta_{t-1}), -\frac{\eta}{C} \sum_{c=1}^{C} \sum_{k=0}^{K-1} \text{desk sk } \nabla_\theta \mathcal{L}_c(\theta_{t-1,c,k}) \rangle \tag{72}$$

$$= -\eta \langle R\nabla_\theta \mathcal{L}(\theta_{t-1}), \frac{1}{C} \sum_{c=1}^{C} \sum_{k=0}^{K-1} R\nabla_\theta \mathcal{L}_c(\theta_{t-1,c,k}) - KR\nabla_\theta \mathcal{L}(\theta_{t-1}) + KR\nabla_\theta \mathcal{L}(\theta_{t-1}) \rangle \tag{73}$$

$$= -\eta K \langle R\nabla_\theta \mathcal{L}(\theta_{t-1}), R\nabla_\theta \mathcal{L}(\theta_{t-1}) \rangle \tag{74}$$

$$- \frac{\eta}{C} \sum_{c=1}^{C} \langle R\nabla_\theta \mathcal{L}(\theta_{t-1}), \sum_{k=0}^{K-1} [R(\nabla_\theta \mathcal{L}_c(\theta_{t-1,k,c}) - R\nabla_\theta \mathcal{L}(\theta_{c,t-1,0}))] \rangle \tag{75}$$

$$\le -\eta K(1 - \varepsilon_1) \|\nabla_\theta \mathcal{L}(\theta_{t-1})\|^2 + \eta \cdot \eta_{\text{local}} \left( c_s \varrho^2 + \frac{c_H c_l}{\sqrt{m}} \right) (1 + \varepsilon_2) K^2 G^2 \tag{76}$$

where,

$$\|\nabla_\theta \mathcal{L}(\theta_{t-1,k,c}) - \nabla_\theta \mathcal{L}(\theta_{t-1,c,0})\| \le \left( c_s \varrho^2 + \frac{c_H c_l}{\sqrt{m}} \right) \|\theta_{t-1,k,c} - \theta_{t-1,c,0}\| \tag{77}$$

$$\le \left( c_s \varrho^2 + \frac{c_H c_l}{\sqrt{m}} \right) \| \sum_{j=0}^{k-1} -\eta_{\text{local}} \nabla_\theta \mathcal{L}(\theta_{t-1,j,c})\| \tag{78}$$

$$\le \eta_{\text{local}} \left( c_s \varrho^2 + \frac{c_H c_l}{\sqrt{m}} \right) KG \tag{79}$$

### D.2.2   Bounding $T_2$

$$\langle \nabla_\theta f(\theta_{t-1}; \mathbf{x}_i), \theta_t - \theta_{t-1} \rangle^2 = \left\langle \nabla_\theta f(\theta_{t-1}; \mathbf{x}_i), -\frac{\eta}{C} \sum_{c=1}^{C} \sum_{k=0}^{K-1} \text{desk sk } \nabla_\theta \mathcal{L}_c(\theta_{c,t-1,k}) \right\rangle^2 \tag{80}$$

$$= \frac{\eta^2}{C^2} \left( \sum_{c=1}^{C} \sum_{k=0}^{K-1} \langle \nabla_\theta f(\theta_{t-1}; \mathbf{x}_i), \text{desk sk } \nabla_\theta \mathcal{L}_c(\theta_{c,t-1,k}) \rangle \right)^2 \tag{81}$$

$$\le \eta^2 (1 + \varepsilon_3)^2 \varrho^2 K^2 G^2 \tag{82}$$

for some suitable choice of $\varepsilon_2$ and $b$ which we set later. Thus,

$$\frac{1}{2C} \sum_{c=1}^{C} \left( \frac{1}{n_c} \sum_{i=1}^{n_c} \ell_{i,c}^{''} \langle \nabla_\theta f(\theta_{t-1}; \mathbf{x}_i), \theta_t - \theta_{t-1} \rangle^2 \right) \le \frac{\eta^2}{2} c_s \left( 1 + \varepsilon_3 \right)^2 \varrho^2 K^2 G^2 \qquad (83)$$

### D.2.3 Bounding $T_3$

Using eigen-decomposition,

$$(\theta_t - \theta_{t-1})^T \nabla_\theta^2 H_{i,c}(\theta_t - \theta_{t-1}) = \sum_{j=1}^{p} \Lambda_{j,i,c} \langle \theta_t - \theta_{t-1}, \mathbf{v}_{j,i,c} \rangle^2 \qquad (84)$$

where $\Lambda_{j,i,c}, \mathbf{v}_{j,i,c}$ for $j \in [1, \cdots, p]$ are the eigen-values and eigen-vectors of $H_{i,c} = \ell_i' \nabla_\theta^2 f(\theta_{t-1}; \mathbf{x}_i)$ respectively.

$$\langle \theta_t - \theta_{t-1}, \mathbf{v}_{j,i,c} \rangle^2 = \eta^2 \langle \frac{1}{C} \sum_{c=1}^{C} \sum_{k=0}^{K-1} \text{desk sk } \nabla_\theta \mathcal{L}_c(\theta_{c,t-1,k}), \mathbf{v}_{j,i,c} \rangle^2 \qquad (85)$$

$$\le \eta^2 \langle \frac{1}{C} \sum_{c=1}^{C} \sum_{k=0}^{K-1} \nabla_\theta \mathcal{L}_c(\theta_{c,t-1,k}), \mathbf{v}_{j,i,c} \rangle^2 + \eta^2 \varepsilon_4^2 \| \frac{1}{C} \sum_{c=1}^{C} \sum_{k=0}^{K-1} \nabla_\theta \mathcal{L}_c(\theta_{c,t-1,k}) \|^2 \qquad (86)$$

$$+ 2\varepsilon_4 \eta^2 \| \frac{1}{C} \sum_{c=1}^{C} \sum_{k=0}^{K-1} \nabla_\theta \mathcal{L}_c(\theta_{c,t-1,k}) \|^2 \qquad (87)$$

$$\le \eta^2 \left\langle \frac{1}{C} \sum_{c=1}^{C} \sum_{k=0}^{K-1} \nabla_\theta \mathcal{L}_c(\theta_{c,t-1,k}), \mathbf{v}_{j,i,c} \right\rangle^2 + \eta^2 (\varepsilon_4^2 + 2\varepsilon_4) K^2 G^2 \qquad (88)$$

for suitable choice of $\varepsilon_4$ and $b$.

$$\frac{1}{2C} \sum_{c=1}^{C} \left( \frac{1}{n_c} \sum_{i=1}^{n_c} \left( \sum_{j=1}^{p} \Lambda_{j,i,c} \langle \theta_t - \theta_{t-1}, \mathbf{v}_{j,i,c} \rangle^2 \right) \right) \qquad (89)$$

$$\le \frac{1}{2C} \sum_{c=1}^{C} \left( \frac{1}{n_c} \sum_{i=1}^{n_c} \left( \sum_{j=1}^{p} \Lambda_{j,i,c} \left\{ \left\langle \frac{1}{C} \sum_{c=1}^{C} \sum_{k=0}^{K-1} \nabla_\theta \mathcal{L}_c(\theta_{c,t-1,k}), \mathbf{v}_{j,i,c} \right\rangle^2 + (\varepsilon_4^2 + 2\varepsilon_4) K^2 G^2 \right\} \right) \right) \qquad (90)$$

$$\le \frac{\eta^2}{2} \frac{c_l c_H}{\sqrt{m}} K^2 G^2 + \frac{\eta^2}{2} \left( 2\varepsilon_4 + \varepsilon_4^2 \right) \frac{c_l \kappa c_H}{\sqrt{m}} K^2 G^2 \qquad (91)$$

### D.2.4 Combining $T_1, T_2, T_3$:

Combining $T_1, T_2$ and $T_3$ we get,

$$\mathcal{L}(\theta_t) \le \mathcal{L}(\theta_{t-1}) - \eta K(1 - \varepsilon_1) \|\nabla_\theta \mathcal{L}(\theta_{t-1})\|^2 + \eta \cdot \eta_{\text{local}} \left( c_s \varrho^2 + \frac{c_H c_l}{\sqrt{m}} \right) (1 + \varepsilon_2) K^2 G^2$$

$$+ \frac{\eta^2}{2} c_s \left( 1 + \varepsilon_3 \right)^2 \varrho^2 K^2 G^2 + \frac{\eta^2}{2} \frac{c_l c_H}{\sqrt{m}} K^2 G^2 + \frac{\eta^2}{2} \left( 2\varepsilon_4 + \varepsilon_4^2 \right) \frac{c_l \kappa c_H}{\sqrt{m}} K^2 G^2 \qquad (92)$$

$$\le \mathcal{L}(\theta_{t-1}) - \eta K(1 - \varepsilon_1) \|\nabla_\theta \mathcal{L}(\theta_{t-1})\|_2^2$$

$$+ \eta \left[ \eta_{\text{local}} \left( c_s \varrho^2 + \frac{c_H c_l}{\sqrt{m}} \right) (1 + \varepsilon_2) + \frac{\eta}{2} c_s \left( 1 + \varepsilon_3 \right)^2 \varrho^2 + \frac{\eta}{2} \frac{c_l c_H}{\sqrt{m}} + \frac{\eta}{2} \left( 2\varepsilon_4 + \varepsilon_4^2 \right) \frac{c_l \kappa c_H}{\sqrt{m}} \right] K^2 G^2 \qquad (93)$$

Using PL condition and iterating over this recursion we have:

$$\mathcal{L}(\theta_T) - \mathcal{L}(\theta^*) \le (1 - 2\mu\eta K(1-\varepsilon_1))^T (\mathcal{L}(\theta_0) - \mathcal{L}(\theta^*)) + \eta^2 C_2(\varepsilon, m, \kappa) \frac{1 - (1 - 2\mu\eta K(1-\varepsilon_1))^T}{1 - (1 - 2\mu\eta K(1-\varepsilon_1))}$$
(94)

$$\le (\mathcal{L}(\theta_0) - \mathcal{L}(\theta^*)) e^{-2(1-\varepsilon)\mu\eta KT} + \frac{\eta C_2(\varepsilon, m, \kappa) K G^2}{2\mu}$$
(95)

where $\eta \le \frac{1}{2\mu K(1-\varepsilon)}$. For simplicity, let $\eta_{\text{global}} = 1$ i.e. $\eta = \eta_{\text{local}}$ and $\varepsilon_1 = \varepsilon_2 = \varepsilon_3 = \varepsilon$. And,

$$C_2(\varepsilon, m, \kappa) = \frac{1}{1-\varepsilon} \left[ \left( c_s \varrho^2 + \frac{c_H c_l}{\sqrt{m}} \right)(1+\varepsilon) + \frac{1}{2} c_s (1+\varepsilon)^2 \varrho^2 + \frac{c_l c_H}{2\sqrt{m}} + \frac{1}{2}\left(2\varepsilon + \varepsilon^2\right) \frac{c_l \kappa \mathbf{c}_H}{\sqrt{m}} \right]$$
(96)

$$C_2(\varepsilon, m, \kappa) = \mathcal{O}(\varrho^2 + \frac{c_H}{\sqrt{m}}) + \mathcal{O}(\frac{\varepsilon\kappa\mathbf{c}_H}{\sqrt{m}})$$
(97)

### D.2.5 Choice of sketching dimension $b$:

Using A.1, for each step $t \in [0, \cdots T]$, union bounding over $T_1, T_2$ and $T_3$, Union bounding over all T time-steps, we get

$$b = \Omega\left( \frac{1}{\varepsilon^2} \log^3(p^2 NT |C| \|K| / \delta) \right)$$
(98)

where $N = \sum_{i=1}^{C} n_c$ is the number of training samples. $\qquad \square$

## E  Communication Efficiency

**Theorem E.1.** If Theorem 4.2 holds, then with $\tilde{\mathcal{O}}\left( C \max\left\{ 1, \frac{C_2(\varepsilon, m, \kappa)G^2}{2\mu^2(1-\varepsilon)K\epsilon} \right\} \log\left( \frac{\mathcal{L}(\theta_0) - \mathcal{L}(\theta^*)}{\epsilon} \right) \right)$ bits of communication, with w.p at least $1 - \delta$ Algorithm: 1 outputs an $\epsilon$-optimal solution $\theta_T$ satisfying:

$$\mathcal{L}(\theta_T) - \mathcal{L}(\theta^*) \le \epsilon$$
(99)

*Proof.* Setting $\eta = \min\{\frac{1}{2\mu K(1-\varepsilon)}, \frac{\epsilon\mu}{C_2(\varepsilon, m, \kappa)KG^2}\}$ and $T = \frac{1}{2\mu\eta K(1-\varepsilon)} \log\left( \frac{2(\mathcal{L}(\theta_0) - \mathcal{L}(\theta^*))}{\epsilon} \right)$, we have:

$$\exp(-2\mu\eta K(1-\varepsilon)T)(\mathcal{L}(\theta_0) - \mathcal{L}(\theta^*)) \le \frac{\epsilon}{2}$$
(100)

$$\frac{\eta C_2(\varepsilon, m, \kappa) K G^2}{2\mu} \le \frac{\epsilon}{2}$$
(101)

$$\implies (\mathcal{L}(\theta_t) - \mathcal{L}(\theta^*)) \le (1 - 2\mu\eta K(1-\varepsilon))^T (\mathcal{L}(\theta_0) - \mathcal{L}(\theta^*)) + \frac{\eta C_2(\varepsilon, m, \kappa) K G^2}{2\mu} \le \epsilon$$
(102)

The total number of communication bits are given as:

$$CbT = \tilde{\mathcal{O}}\left( C \max\left\{ 1, \frac{C_2(\varepsilon, m, \kappa)G^2}{2\mu^2(1-\varepsilon)\epsilon} \right\} \log\left( \frac{\mathcal{L}(\theta_0) - \mathcal{L}(\theta^*)}{\epsilon} \right) \right)$$
(103)

where, $\tilde{\mathcal{O}}$ hides poly-log dependence. $\qquad \square$

## F  Additional Experimental Details & Results

We provide the overview of the local `Top-r` baseline in Algorithm 2.

**Algorithm 2** Local `Top-r`-based Distributed Learning.

---
**Hyperparameters:** server learning rate $\eta_{\text{global}}$, local learning rate $\eta_{\text{local}}$.
**Inputs:** local datasets $\mathcal{D}_c$ of size $n_c$ for clients $c = 1, \ldots, C$, number of communication rounds $T$.
**Output:** final model $\theta_T$.

 1: Broadcast a random `SEED` to the clients.
 2: **for** $t = 1, \ldots, T$ **do**
 3:    **On Client Nodes:**
 4:    **for** $c = 1, \ldots, C$ **do**
 5:      **if** $t = 1$ **then**
 6:        Receive the random `SEED` from the server. Initialize the local model $\theta_{c,1} \in \mathbb{R}^p$ using the random `SEED`.
 7:      **else**
 8:        Receive $\bar{\bar{\Delta}}_{t-1}$ from the server.
 9:        Update the model parameters $\theta_t \leftarrow \theta_{t-1} + \bar{\bar{\Delta}}_{t-1}$.
10:        Assign the local model's parameters $\theta_{c,t} \leftarrow \theta_t$ to be updated locally.
11:      **end if**
12:      **for** $k = 1, \ldots, K$ **do**
13:        $\theta_{c,t} \leftarrow \theta_{c,t} - \eta_{\text{local}} \cdot \nabla_\theta \mathcal{L}_c(\theta_{c,t})$
14:      **end for**
15:      $\Delta_{c,t} \leftarrow \theta_{c,t} - \theta_t$
16:      Send `Top-r` sparsified updates $\hat{\Delta}_{c,t} \leftarrow \text{Top-r}(\Delta_{c,t})$ to the server.
17:    **end for**
18:
19:    **On the Server Node:**
20:    Receive `Top-r` sparsified updates $\hat{\Delta}_{c,t}$ from clients $c = 1, \ldots, C$.
21:    Aggregate them $\bar{\bar{\Delta}}_t \leftarrow \eta_{\text{global}} \cdot \frac{1}{C} \sum_{c=1}^{C} \hat{\Delta}_{c,t}$
22:    Broadcast $\bar{\bar{\Delta}}_t$ to the clients.
23: **end for**

---

# G   Additional Results on Spectral Density of Predictor Hessian

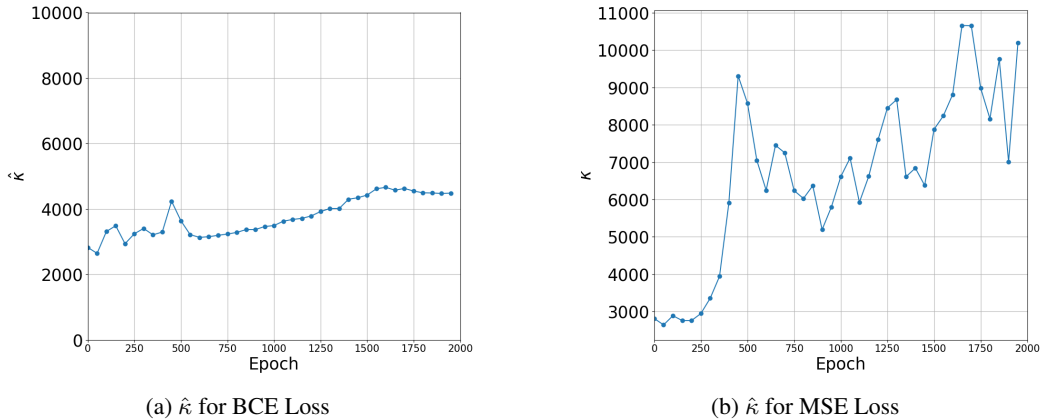

(a) $\hat{\kappa}$ for BCE Loss               (b) $\hat{\kappa}$ for MSE Loss

Figure 2: Estimate $\hat{\kappa}$ of $\kappa = \sum_{i=1}^{p} |\Lambda_i| / \Lambda_{\max}$ over training iterations.

This section presents empirical results verifying our assumption on the sum of absolute eigenvalues of the predictor Hessian (Assumption 4.2). Since it is infeasible to compute all eigenvalues for deep models like ResNet-18, we rely on numerical approximations introduced in several prior works [24, 78]. SLQ is a technique in numerical linear algebra to estimate the ESD of large matrices. The complete algorithm can be found in Algorithm 3. We use PyHessian [24], an open-source implementation of SLQ for our experiments. We keep the default parameters $n_v = 10, m = 100$ for the plots as they have been shown to be of high accuracy. We train a ResNet-18 from scratch model

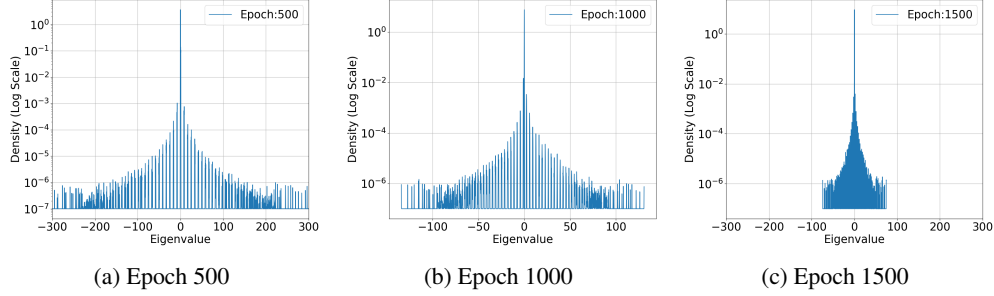

| (a) Epoch 500 | (b) Epoch 1000 | (c) Epoch 1500 |

Figure 3: Spectral Density of Predictor Hessian :$H(\theta, \mathbf{x}_i) = \ell'_i \nabla^2_\theta f(\theta; \mathbf{x}_i)$ for a fixed training input across training epochs. Dataset : 1000 samples from CIFAR-10 dataset. Model: ResNet-18. The model has $1.1 \times 10^7$ parameters. Loss function: Binary Cross Entropy(BCE) Loss.

---

**Algorithm 3** Stochastic Lanczos Quadrature for ESD Computation (Ghorbani et al. [24])

---

1: **Input:** Parameter: $\theta$, degree $m$, and $n_v$.
2: Compute the gradient of $\theta$ by backpropagation, i.e., compute $g_\theta = \frac{df(\theta; x_i)}{d\theta}$.
3: **for** $i = 1, 2, \ldots, n_v$ **do**
4:  {Different Seeds}
5:  Draw a random vector $v$ from $\mathcal{N}(0, 1)$ and normalize it (same dimension as $\theta$).
6:  Get the tridiagonal matrix $T$ through Lanczos algorithm.
7:  Compute $\tau_k^{(i)}$ and $\tilde{\lambda}_k^{(i)}$ from $T$.
8:  $\phi_\sigma^{z_i} = \sum_{k=1}^q \tau_k f(\tilde{\lambda}_k; t, \sigma)$
9: **end for**
10: **Return** $\phi(t) = \frac{1}{n_v} \sum_{l=1}^{n_v} \left( \sum_{i=1}^q \tau_i^{(l)} f(\tilde{\lambda}_i^{(l)}; t, \sigma) \right)$

---

to classify 1000 samples from two classes of CIFAR-10. We use a learning rate of $1e - 3$, SGD as the optimizer and and perform GD. We use a fix training sample to plot the ESD of predictor Hessian $H(\theta; \mathbf{x}_i) = \ell'_i \nabla^2_\theta f(\theta; \mathbf{x}_i)$ for two choices of loss functions : BCE (Binary Cross Entropy) and MSE (Mean Squared Loss). We compute the spectrum by backpropagating through the output layer instead of the loss. From the results in Figure 3, we note that a bulk of eigenvalues are close to zero, and the spectral density decays quickly far from zero, i.e., most of the eigenvalues are tiny. Given the approximate density as a normalized histogram, we compute the empirical average and estimate $\hat{\kappa}$ an approximation to $\kappa = \sum |\Lambda_i| / \Lambda_{\max}$. Our results for $\hat{\kappa}$ are presented in Figure 2.

